# Mixture of Adversarial LoRAs: Boosting Robust Generalization in Meta-Tuning

**Xu Yang**[1]   **Chen Liu**[† 1]   **Ying Wei**[† 2]

[1] City University of Hong Kong   [2] Zhejiang University

xyang337-c@my.cityu.edu.hk   chen.liu@cityu.edu.hk   ying.wei@zju.edu.cn

## Abstract

This paper introduces AMT, an **A**dversarial **M**eta-**T**uning methodology, to boost the robust generalization of pre-trained models in the out-of-domain (OOD) few-shot learning. To address the challenge of transferring knowledge from source domains to unseen target domains, we construct the robust LoRAPool by meta-tuning LoRAs with dual perturbations applied to not only the inputs but also singular values and vectors of the weight matrices at various robustness levels. On top of that, we introduce a simple yet effective test-time merging mechanism to dynamically merge discriminative LoRAs for test-time task customization. Extensive evaluations demonstrate that AMT yields significant improvements, up to 12.92% in clean generalization and up to 49.72% in adversarial generalization, over previous state-of-the-art methods across a diverse range of OOD few-shot image classification tasks on three benchmarks, confirming the effectiveness of our approach to boost the robust generalization of pre-trained models. Our code is available at https://github.com/xyang583/AMT.

## 1 Introduction

Few-shot learning (FSL) has recently been revolutionized by large-scale pre-trained vision transformer models [1, 2, 3, 4, 5]. Their generalization capability can be further enhanced with a few annotated examples, achieving impressive performance across a broad spectrum of downstream tasks [6, 7, 8, 9, 10]. Building on this foundation, meta-tuning emerges as a powerful strategy that integrates the broad generalization capabilities of pre-trained prior knowledge with the adaptive flexibility of meta-learning, allowing models to quickly adapt to new tasks in few-shot scenarios [11, 12].

Despite its success, the robust generalization of meta-tuning to defend against adversarial attacks [13, 14, 15] and adapt to out-of-distribution (OOD) downstream tasks [16, 17, 18] remains an ongoing challenge. However, it is crucial for various real-world applications such as medical imaging diagnostics and autonomous driving to simultaneously achieve competitive performance on adversarial examples or out-of-distribution data. Deployed models often encounter novel environments with distribution shifts between training and test data, including variations in hospital equipment and protocols [19] or diverse urban road scenarios [20]. Moreover, these models are vulnerable to adversarial attacks leading to harmful diagnoses or unsafe driving decisions. For instance, adversaries can perturb sensor signals to deceive 2D or 3D medical imaging models [21], manipulate traffic signs with malicious stickers [22], or fool the autopilot into following unsafe trajectories [23].

In this paper, we delve into leveraging adversarial training and meta-tuning to enhance robust generalization of pre-trained vision transformers across different domains. Compared with previous meta-tuning approaches, this involves two unique aspects. Firstly, when incorporating adversarial

---

[†] Corresponding authors.

examples, the model should learn to adapt to the worst-case tasks while preserving its inherent generalization capabilities. Inspired by the observation that the singular values distribution of weight parameters undergoes significant changes during fine-tuning [24], we aim to explicitly strengthen the principal components of pre-trained model weight matrices during meta-tuning. To this end, we inject perturbations on both input and principal singular values and vectors via the incremental meta-update of the Low-rank Adapter (LoRA) [25, 26] on top of frozen pre-trained parameters. Secondly, the adversarial perturbation needs to simulate wide distribution variations from the training environment, and care must be taken to avoid interference when training with multiple perturbation types [27, 28]. Thus, we introduce an adaptive robust LoRAPool constructed by meta-tuning different LoRAs in parallel for different attack strengths. To adapt to novel tasks from unseen distributions, we view the robust LoRAPool as the basis and integrate meta-updated principal components into the pre-trained model through a test-time merging mechanism for downstream task customization.

Our main contributions are summarized as follows:

- We propose AMT, a novel adversarial meta-tuning approach for enhancing the robust generalization of pre-trained vision transformers across diverse domains.
- By injecting the adversarial perturbations on the inputs, singular values and vectors of the weight matrices, the core components of pre-trained model weights are consolidated for worst-case tasks. We further enhance this approach with the adaptive robust LoRAPool meta-tuned under varying perturbation budgets, without compromising the pre-trained model's inherent capabilities.
- We integrate discriminative principle components into the pre-trained model via a simple yet effective test-time merging mechanism for customizing task-specific feature extractors, which is compatible with other test-time fine-tuning methods.
- We experimentally evaluate our method on challenging large-scale out-of-domain few-shot image classification benchmarks, including Meta-Dataset [16] that consists of 9 OOD datasets, as well as BSCD-FSL [29] and fine-grained datasets [30] comprising another 8 OOD datasets. Our method achieves impressive few-shot performance across domains, significantly outperforming previous state-of-the-art methods in clean generalization by up to 12.92% and in adversarial generalization by up to 49.72%.

## 2 Related work

**Out-of-Domain Few-shot Learning and Meta-Learning.** Out-of-Domain Few-Shot Learning (OOD-FSL) aims to transfer prior knowledge learned on source domains to unseen target domains to address the few-shot learning problem [16, 17, 18, 29, 31, 32, 33, 34, 35, 36]. Meta-learning relies on episodic training to learn parameter initialization [37, 38, 39], optimization rule [40, 41, 42] or a transferable metric space [43, 44, 45, 46, 47] as prior knowledge for quick adaptation to new tasks. To tackle distribution shifts, many methods are proposed by building a universal feature representation with multiple feature extractors [32, 31], conditioning batch normalization parameters [48, 30, 33], or test-time gradient-based fine-tuning [34]. Most related to our work is FLUTE [33], which jointly trains the feature extractor with multiple sets of Feature-wise Linear Modulation (FiLM) [49] parameters on multiple training datasets and combines them as the initialization for gradient descent at test time. Our method AMT stands in the single source domain setting and differs from previous works in that our adversarial meta-tuning does not compromise the pre-trained model, and the adaptive merging mechanism of the robust LoRAPool performs task customization in a non-parametric manner without the requirement of gradient descent, ensuring scalability with newly added components to the pool.

**Vision Transformers in Few-shot Learning**. Vision Transformers (ViTs) have gained prominence as the foundation model due to their ability to capture long-range dependencies in data [50, 51, 52]. Self-supervised pre-training effectively endows vision transformer with data-driven and well-generalized prior [1, 2, 53, 54], especially for the few-shot learning task. In the spirit of transfer learning, one line of works leverages self-distillation framework to seek universal feature representations without meta-training [55, 56, 57] and directly learns auxiliary visual prompts [58] and attention scaling matrices [59] on the support set through gradient descent during meta-testing. Another important research direction is developing meta-learning techniques to enhance pre-trained models with input-conditioned prompts [8] and task-specific masks [11]. PMF [12] contributes a strong baseline by meta-tuning the full model. In this work, we also ground our method on pre-trained vision transformers and show that adversarial meta-tuning can further boost their robust generalization across downstream tasks. Also, our contribution is orthogonal to other existing test-time fine-tuning methods and provides a better starting point to improve their performance at test time.

**Adversarial Training for Out-of-Distribution Generalization.** Adversarial training [13] is one of the most effective defense techniques to improve the model adversarial robustness by minimizing a locally maximized loss function via adversarial perturbation on inputs [13, 60, 61, 62, 63, 64, 65] and model parameters [66]. Despite widely recognized trade-offs between adversarial robustness and clean accuracy [67, 68], and between in-distribution (ID) and out-of-distribution (OOD) generalization [63, 69], there exist strategies to achieve better balances among these trade-offs. These strategies include modified adversarial training regime [65], dual sets of parameters [70, 71, 72], model ensemble [73], multi-scale patch perturbations [74], or partial fine-tuning strategy [75]. Furthermore, since adversarially perturbed input data can be viewed as a special type of OOD data [76], recent studies [77, 78, 79] have demonstrated that adversarial pre-training can enhance generalization performance on downstream datasets and improve robustness against distribution shifts. Compared with sample-wise adversarial attacks, where all samples in each domain share the universal perturbation, the distributional attacks in a low-rank structure show the capability of making the models resistant against adversarial perturbations of higher magnitude [80, 81] Inspired yet different from the previous attack methods, our method utilizes a mixture of adversarial low-rank adaptors customized for meta-tuning to enhance the robust generalization of clean pre-trained models.

**Adversarial Meta-Learning.** There is a series of works that leverage adversarial training to enhance the few-shot learner's adversarial robustness [14, 15, 82, 83]. However, compared with standard few-shot learning, the adversarially trained model has degraded clean accuracy [14]. Adversarial training is also utilized to improve the cross-domain few-shot learning performance by attacking individual image pixels [84] and features [85, 86]. For example, StyleAdv [86] perturbs each sample style in a task through attacking statistical information of AdaIN [87] and updating all parameters. Our approach diverges from these existing methods, aiming to further enhance the generalization performance of large-scaled pre-trained models. To achieve this, we propose to inject double perturbations on inputs as well as singular values and vectors over the entire query set as a whole during meta-tuning, while keeping all pre-trained parameters frozen to preserve prior knowledge.

**Parameter-Efficient Few-Shot Learning.** To reduce the computational cost associated with full-model fine-tuning, various parameter-efficient fine-tuning (PEFT) methods have been proposed that only update a small set of parameters, including inserting soft prompts [88, 89], adding adapter modules [90, 91, 92], and introducing low-rank matrices [25, 93, 94]. Recent works have shown that PEFT achieves comparable or superior performance than standard fine-tuning in the few-shot setting for large language models [95]. In this work, we explore crafting the small parameter sets via meta-tuning to boost robust generalization of pre-trained vision transformers. Concretely, we leverage LoRA [25] as the core parameter-efficient component for constructing the adaptive robust pool, as it enables low-rank updates to be merged into network weights without additional computational or memory costs incurred during inference.

## 3 Problem Formulation

In this work, we focus on out-of-domain few-shot image classification where our goal is to find parameters $\theta$ that generalize well on unseen target domains with the single-source training domain. In this context, the model not only needs to learn novel concepts from limited data but also to generalize well across diverse domains. For each domain, there exists a dataset collected from that environment. During training, we only have access to the single source training dataset $\mathcal{D}_{tr}^{\text{seen}}$, from which each task $\mathcal{T} = (\mathcal{S}, \mathcal{Q})$ is randomly sampled as the input. The support set $\mathcal{S}$ contains $K$ annotated images for each of the $N$ categories: $\mathcal{S} = \{x_s, y_s\}_{s=1}^{NK}$, while the query set $\mathcal{Q}$ contains $M$ images $\mathcal{Q} = \{x_q, y_q\}_{q=1}^{M}$. At evaluation time, the aim is to tackle tasks with novel classes sampled from previously unseen datasets $\mathcal{D}_{test}^{\text{unseen}}$.

## 4 Methods

We introduce our approach in this section. The overall framework of our AMT is illustrated in Figure 1. It contains two components: (i) adversarial singular value and vector perturbation, which explicitly perturbs the singular values and vectors to highlight the principal components in the worst-case tasks; (ii) Adaptive robust LoRAPool, which consists of several adversarially meta-tuned LoRA modules and test-time merging mechanism to adaptively merge them for task customization.

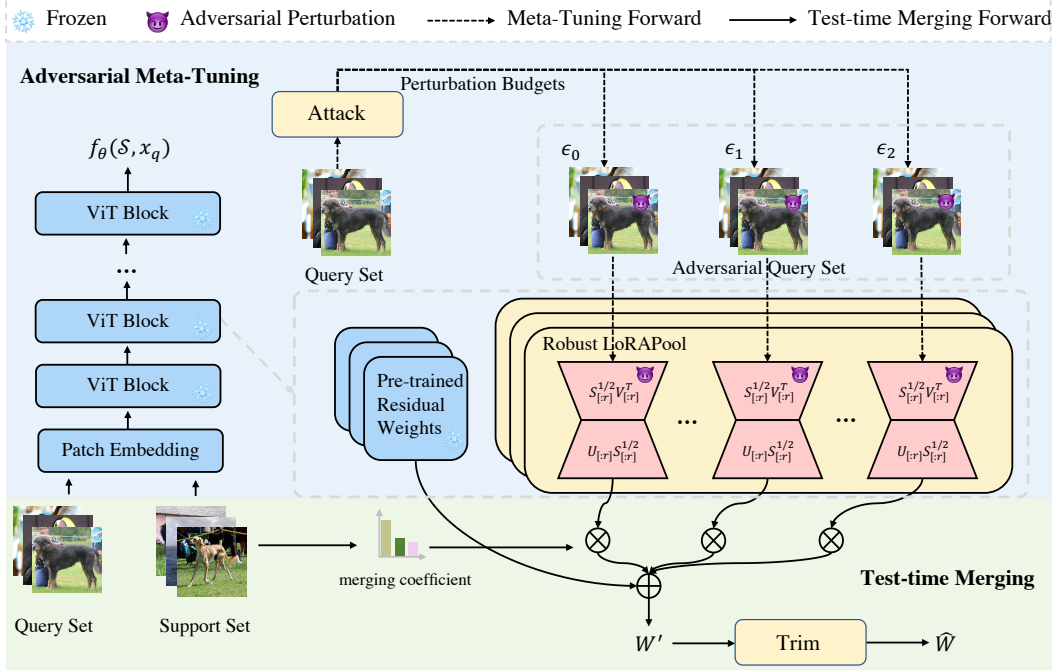

Figure 1: **Overview of our method.** Adversarial perturbations, bounded by different budgets $\epsilon$, are incorporated into the clean query set. To construct the robust LoRAPool, the LoRA modules initialized with SVD results are meta-tuned on the adversarial examples, upon which adversarial perturbations are injected into singular values and vectors. The discriminative incremental updates of principal components are adaptively merged into the pre-trained weights for test-time task customization.

## 4.1 Preliminaries

**Adversarial Meta-Tuning**. We ground our method on a large-scale pre-trained Vision Transformer [50] and then meta-tune the model in an episodic manner [43], following PMF [12]. To robustify the learned meta-knowledge, adversarial meta-tuning adopts the worst-case optimization by injecting the adversarial perturbation $\delta$ to the query image $x_q$ through the minimax strategy [14, 15]. The intuition here is to make the meta-tuned model have the same prediction in the worst-case task.

We consider the $l_\infty$ norm bounded perturbations in this work, so the corresponding optimization problem can be formulated as $\min_\theta \max_{\|\delta\|_\infty \leq \epsilon} \mathcal{L}\left(f_\theta\left(\mathcal{S}, x_q + \delta\right), y_q\right)$ where $f_\theta$ denotes predicted logits of a query example with the model parameters $\theta$, and $\mathcal{L}$ is the meta-task loss, which is usually the cross-entropy loss for image classification. The inner maximization problem can be efficiently solved by gradient-based methods. In practice, Projected Gradient Descent (PGD) [13] is the most popular method to generate adversarial perturbations $\delta$. Specifically, when the step size is $\alpha$, PGD optimizes $\delta$ by running the following update rule for multiple iterations. Here, $\Pi$ is the projection operator to clip $\delta$ so that $\|\delta\|_\infty \leq \epsilon$.

$$\delta \leftarrow \Pi_\epsilon\left(\delta + \alpha \cdot \text{sign}\left(\nabla_\delta \mathcal{L}\left(f_\theta(\mathcal{S}, x_q + \delta), y_q\right)\right)\right). \tag{1}$$

**Low Rank Adaptation**. LoRA [25] is one of the popular parameter-efficient fine-tuning approaches for transformer models. Given a pre-trained weight matrix $W \in \mathbb{R}^{d_{in} \times d_{out}}$, LoRA approximates incremental updates to the parameter matrix with a low-rank decomposition $\triangle W = AB$, where $A \in \mathbb{R}^{d_{in} \times r}$ and $B \in \mathbb{R}^{r \times d_{out}}$, and the rank $r \ll \min(d_{in}, d_{out})$. The LoRA approach can be applied to all the linear layers in the vision transformer. For an input $x$ and a hidden state $h = Wx$, LoRA modifies forward process as $h = (W + \triangle W)x = Wx + ABx$. When fine-tuning, $W$ is frozen while $A$ and $B$ are trainable. In addition, $A$ is randomly initialized via Gaussian initialization while $B$ is initialized to zero, resulting in the incremental update $AB = 0$ at the beginning.

## 4.2 Adversarial Singular Value and Vector Perturbation

Drawing inspiration from the insight that the distribution of singular values undergoes significant changes during fine-tuning [24], we aim to explicitly strengthen the principal components of pre-trained model weight matrices to enhance the model's generalization capability across diverse

target domains. Using the on-the-fly generated adversarial query samples, we inject the worst-case perturbation on singular values and vectors over the entire query set. However, meta-tuning the full model and performing multiple singular value decomposition (SVD) during training are computationally expensive. To this end, we adopt the LoRA formulation to update model parameters during meta-tuning and initialize the incremental updates of LoRA with the result of SVD of weight matrices for the multi-head self-attention (MHA) layer and feed-forward network (FFN) layer in the vision transformer [26].

Formally, for a weight matrix $W \in \mathbb{R}^{d_{in} \times d_{out}}$ and its singular value decomposition $W = U \operatorname{diag}(S) V^T$, where $U \in \mathbb{R}^{d_{in} \times \min(d_{in}, d_{out})}$, $V \in \mathbb{R}^{d_{out} \times \min(d_{in}, d_{out})}$ and $S \in \mathbb{R}^{\min(d_{in}, d_{out})}$ represent the left/right singular vectors and the singular values in descending order, respectively. In the LoRA formulation $\triangle W = AB$ with the rank $r$, the top $r$ singular values and corresponding vectors are utilized to initialize $A \in \mathbb{R}^{d_{in} \times r}$ and $B \in \mathbb{R}^{r \times d_{out}}$, while the residual singular values and vectors are used to calculate the residual matrix $W^{res} \in \mathbb{R}^{d_{in} \times d_{out}}$ for error correction:

$$
\begin{aligned}
A &= U_{[:r]} \operatorname{diag}\left(S_{[:r]}^{1/2}\right) \in \mathbb{R}^{d_{in} \times r} \\
B &= \operatorname{diag}\left(S_{[:r]}^{1/2}\right) V_{[:r]}^T \in \mathbb{R}^{r \times d_{out}} \\
W^{res} &= U_{[r:]} \operatorname{diag}\left(S_{[r:]}\right) V_{[r:]}^T \in \mathbb{R}^{d_{in} \times d_{out}}
\end{aligned}
\tag{2}
$$

In Equation (2), we have $W = W^{res} + AB$. During training, $W^{res}$ is kept frozen, so the updates of $A$ and $B$ in the subspace approximate the modification of principle singular value and vectors.

To boost the generalization performance of the model, we utilize the sharpness-aware minimization (SAM) [96] to update $A$ and $B$. Specifically, we find the worst-case perturbation $\delta_A$ and $\delta_B$ in the neighborhood of $A$ and $B$ by gradient ascent. $\delta_A$ is calculated by the following equation where $M$ is the size of query set and $\eta_1$ depicts the size of the neighbourhood. $\delta_B$ can be calculated similarly.

$$
\delta_A = \eta_1 \cdot \frac{1}{M} \sum_{q=1}^{M} \nabla_A \mathcal{L}(f_{W^{res}+AB}(\mathcal{S}, x_q^{adv}), y_q))
\tag{3}
$$

Here, we omit other parameters in $\theta$ for notation simplicity. We then use the gradient based on the worst-case neighborhood to update $A$ and $B$. Given the learning rate $\eta_2$, the update rule for $A$ is as follows. $B$ is updated similarly using the same learning rate.

$$
A \leftarrow A - \eta_2 \cdot \frac{1}{M} \sum_{q=1}^{M} \nabla_A \mathcal{L}(f_{W^{res}+(A+\delta_A)B}(\mathcal{S}, x_q^{adv}), y_q))
\tag{4}
$$

Different from prior adversarial meta-learning works [14, 15], this paper focuses on improving both the clean accuracy and cross-domain robustness for few-shot learning [63]. The meta-objective function of AMT is the combination of both aspects:

$$
\mathcal{L} = \mathcal{L}_{CE}\left(f_{W^{res}+AB}\left(\mathcal{S}, x_q\right), y_q\right) + \lambda_{adv} D_{\mathrm{KL}}\left(f_{W^{res}+AB}\left(\mathcal{S}, x_q^{adv}\right) \| f_{W^{res}+AB}\left(\mathcal{S}, x_q\right)\right)
\tag{5}
$$

where $\mathcal{L}_{CE}$ is the original cross-entropy loss, $D_{\mathrm{KL}}$ is the Kullback-Leibler divergence and $\lambda_{adv}$ is the trade-off coefficient. Note that here we use few-shot task loss instead of global classification loss in StyleAdv [86] to generate the adversarial attacks, by which we leverage label randomness to avoid the potential performance degradation caused by true label leaking effect [62].

### 4.3 Adaptive Robust LoRAPool

**Robust LoRAPool Construction**. To simulate various distributional shifts for the unseen tasks, we adversarially meta-tune $P$ LoRA modules in parallel by Equation (4), each corresponding to a different robustness level controlled by the size of the adversarial budget, i.e., $\epsilon$ in Equation 1. Therefore, we will obtain a robust LoRAPool composed of $P$ LoRA modules $\phi = [A_1 B_1, \ldots, A_P B_P]$. Algorithm 1 shows our adversarial meta-tuning pipeline.

**Test-time Merging**. Given several LoRA modules, the challenge in the evaluation time is to adaptively merge these modules in robust LoRAPool into the pre-trained model to fit the new tasks. It is commonly assumed in domain generalization that unseen distributions fall within the convex hull of the training environments [97, 98], so we consider the LoRAs in the pool as the bases and learn a convex combination adapted to the task at hand.

---

**Algorithm 1** Robust LoRAPools

---

1: **Input**: Source training domain $\mathcal{D}_{tr}^{seen}$; pre-trained weight residual matrix $W^{res}$; $P$ sets of attack configuration candidates;

2: **Output**: Adversarially meta-trained LoRAPool;

3: Initialize adversarial LoRAPool: $\phi = \{\}$

4: **for** $i = 1$ to $P$ (in parallel) **do**

5:      Sample the $i$-th set of $\epsilon_i$, $\alpha_i$ from attack configuration candidates.

6:      Initialize the LoRA parameter $AB$ via Eq. (2);

7:      **while** not converged **do**

8:          Sample a task $\mathcal{T} = \{\mathcal{S}, \mathcal{Q}\} \sim \mathcal{D}_{tr}^{seen}$.

9:          Generate adversarial query set $\mathcal{Q}_{adv} = \{x_q^{adv}, y_q\}_{q=1}^{M}$ with $\epsilon_i$, $\alpha_i$ via Eq. (1)

10:          // Perturb singular value and vectors

11:          $\delta_A = \eta_1 \cdot \frac{1}{M} \sum_{q=1}^{M} \nabla_A \mathcal{L}(f_{W^{res}+AB}(\mathcal{S}, x_q^{adv}), y_q))$

12:          $\delta_B = \eta_1 \cdot \frac{1}{M} \sum_{q=1}^{M} \nabla_B \mathcal{L}(f_{W^{res}+AB}(\mathcal{S}, x_q^{adv}), y_q))$

13:          // Update $AB$ via SGD

14:          $A \leftarrow A - \eta_2 \cdot \frac{1}{M} \sum_{q=1}^{M} \nabla_A \mathcal{L}(f_{W^{res}+(A+\delta_A)B}(\mathcal{S}, x_q^{adv}), y_q))$

15:          $B \leftarrow B - \eta_2 \cdot \frac{1}{M} \sum_{q=1}^{M} \nabla_B \mathcal{L}(f_{W^{res}+A(B+\delta_B)}(\mathcal{S}, x_q^{adv}), y_q))$

16:      **end while**

17:      $\phi = \phi \bigcup AB$

18: **end for**

---

To estimate the coefficient of this combination, we propose blending intra-class compactness and inter-class divergence on the support set as the criterion to extract the most discriminative features for classification. To reduce the computational cost of calculating pair-wise similarity between all support samples, we leverage the class prototype to approximate the cluster center of each class and calculate sample-prototype distances. Formally, for the $i$-th LoRA in the pool and the $c$-th class out of the total $N$ classes, we denote the class prototype as the average of per-class support features $\mathbf{p}_{i,c} = \frac{1}{K} \sum_{y_s=c} \mathbf{f}_{W^{res}+A_i B_i}(x_s)$. The intra-class compactness $C_i$ and the inter-class divergence $V_i$ are then respectively defined as,

$$C_i = \frac{1}{NK} \sum_{s=1}^{NK} \gamma\left(\mathbf{f}_{W^{res}+A_i B_i}(x_s), \mathbf{p}_{i,y_s}\right), \quad V_i = \frac{1}{NK} \sum_{s=1}^{K} \sum_{\substack{c=1 \\ c \neq y_s}}^{N} \gamma\left(\mathbf{f}_{W^{res}+A_i B_i}(x_s), \mathbf{p}_{i,c}\right) \quad (6)$$

where $\gamma(\cdot, \cdot)$ denotes the cosine similarity between two feature vectors. After calculating the intra-class compactness and inter-class divergence for each LoRA, the merging coefficient $\zeta_i$ for each LoRA module can be estimated as

$$\zeta_i = \frac{\text{Top}_k\left(\exp\left(-\beta(1 - (\lambda C - (1-\lambda)V))\right)\right)_i}{\sum_{i=1}^{k} \text{Top}_k\left(\exp\left(-\beta(1 - (\lambda C - (1-\lambda)V))\right)\right)_i} \quad (7)$$

where $\beta$ and $\lambda$ stand for smooth and balance factors, respectively. The operation $\text{Top}_k$ before softmax refers to selecting the top $k$ LoRA modules with the largest score and the rest LoRAs are deactivated for the current task. The merged weight matrix is then calculated as $W' = W^{res} + \sum_{i=1}^{P} \zeta_i A_i B_i$. To address the issue of interference stemming from redundant components during merging [99], we introduce singular value trimming, retaining only the largest top-$\rho\%$ singular values and resetting the rest to zero to obtain the final task-specific weight $\widehat{W}$:

$$\widehat{W} = \text{trim}(W') \quad (8)$$

This design provides high expressiveness and flexibility by specifying suitable LoRAs for novel tasks, significantly enhancing the model's adaptation ability to generalize across unseen domains. Algorithm 2 in the Appendix A shows our test-time merging algorithm pipeline.

**Network Inference**. After obtaining the task-specific feature extractor through test-time merging, we can employ it directly for inference and perform the nearest-centroid classification [44, 12]. To further improve the few-shot performance in each novel task, our AMT is compatible with other cutting-edge test-time fine-tuning approaches, and thus we introduce a variant AMT-FT, which allows for additional full [12] or efficient [59] fine-tuning.

# 5 Experiments

We evaluate the effectiveness of the proposed AMT on three cross-domain few-shot image classification benchmarks in Section 5.1. Additionally, we present ablation studies in Section 5.2, conduct a broader analysis in Section 5.3, and compare our approach with other PEFT methods in Section 5.4.

**Experimental setup**. We evaluate AMT using the large-scale cross-domain few-shot classification benchmarks Meta-Dataset [16], BSCD-FSL [29] and fine-grained datasets [30].Note that, in the main experiments, all methods utilize a single model trained on the source domain ImageNet to analyze the trade-offs between robustness and generalization. The details of each benchmark are described in Appendix B.1. And training and evaluation details are included in Appendix B.2. We conduct a comprehensive hyperparameter study in Appendix H.

**Baselines**. We adopt the state-of-the-art PMF [12] as the meta-tuning baseline method and use ATTNSCALE [59] as the baseline for an efficient test-time fine-tuning approach. To evaluate our approach against previous adversarial few-shot learning methods, we choose StyleAdv [86] as the representative. All methods employ a Vision Transformer [50] which is DINO-pretrained [1] on ImageNet-1K in our main experiments.

## 5.1 Comparison with State-Of-The-Art Methods

**Clean OOD-FSL**. In Table 1, we evaluate AMT on Meta-Dataset to investigate its generalization performance on OOD few-shot learning problem in both the 5-way 1-shot and 5-shot settings. We group approaches in two settings. The tuning-free setting does not involve additional training on the support set. We adaptively merge meta-tuned LoRA into pre-trained models via our non-parametric test-time merging mechanism and perform prototype-based classification. Aside from this, the test-time fine-tuning setting allows for training on the support set according to different fine-tuning methods, such as fine-tuning full parameters [12] or partial parameters [59]. Our proposed method AMT consistently achieves superior performance across all domains in the tuning-free setting, up to **12.92%** on Omniglot, compared with previous state-of-the-art methods. Moreover, thanks to the flexible design of LoRAPool and the meta-learned well-generalized initialization for pre-trained models, AMT demonstrates strong compatibility with advanced fine-tuning approaches, further boosting few-shot learning performance, with the improvements of **3.92%** and **4.3%** over PMF [12] and ATTNSCALE [59], respectively. Notably, unlike previous StyleAdv [86], our robust generalization improvement does not sacrifice the in-domain clean accuracy. We attribute this to our adaptive robust LoRAPool design, which completely inherits the pre-trained knowledge and performs customization by injecting discriminative information for unseen tasks. We take a further comparative analysis on BSCD-FSL [29] and fine-grained dataset [30] in Table 2 and under the variable-way-variable-shot setting in Table 18 of Appendix K. The overall performance improvement demonstrates the effectiveness of our method.

**Adversarial OOD-FSL**. We evaluate adversarial robustness under distribution shifts for previous state-of-the-art methods using the PGD-10 attack [13] in Table 3. We observe that the naturally trained meta-tuning method PM [12] is not adversarially robust. The style adversarial attack method StyleAdv [86] is also highly vulnerable to adversarial attacks in most domains and sacrifices nearly seven percentage points in-domain performance. In contrast, our method AMT consistently outperforms previous state-of-the-art methods by a wide margin in terms of both in-domain and out-of-domain robust accuracy, achieving up to **49.72%** on Omniglot. Additionally, our method AMT-FT exhibits synergy with the adversarial test-time fine-tuning strategy, further boosting the in-domain and out-of-domain few-shot adversarial robustness. To take a step further, we measure adversarial robustness against AutoAttack [100] and unseen attacks under distribution shifts in Table 19 and Table 20 of Appendix M, respectively. The results indicate that AMT consistently boosts adversarial generalization across domains. Intriguingly, as shown in Figure 4 of Appendix L, AMT can also handle natural corruptions under distribution shifts. As a whole, AMT improves the trade-offs between adversarial robustness and clean accuracy [68, 63], as well as between ID and OOD generalization [3].

## 5.2 Ablation Study

**Component Analysis**. In Table 4, we demonstrate the effectiveness of various components in our method: adversarial perturbation on query images and singular values and vectors, robust LoRAPool, test-time merging, and singular value trimming. For the method incorporating adversarial perturbation

Table 1: **Few-shot classification clean accuracy (%) on Meta-Dataset benchmark [16] in the** 5-**way** 1-**shot and** 5-**shot settings.** We report the average accuracy in each domain for all methods. TTF: test-time fine-tuning, Avg.: Average. **Bold** entries indicate the best for each task configuration.

| 1-shot | Backbone | TTF | In-domain ImageNet | Out-of-domain Omglot | Acraft | CUB | DTD | QDraw | Fungi | Flower | Sign | COCO | Avg. |
|---|---|---|---|---|---|---|---|---|---|---|---|---|---|---|
| PM [12] | ViT-small | - | 65.07 | 59.03 | 38.13 | 76.18 | 61.56 | 57.29 | 56.03 | 80.41 | 55.17 | 54.42 | 60.35 |
| StyleAdv [86] | ViT-small | - | 56.10 | 62.25 | 40.38 | 66.62 | 55.94 | 57.93 | 53.19 | 81.10 | 54.20 | 48.08 | 57.58 |
| AMT | ViT-small | - | **68.80** | **71.95** | **42.90** | **79.95** | **62.99** | **59.62** | **59.06** | **85.37** | **63.78** | **57.14** | **65.16** |
| PMF [12] | ViT-small | Y | 65.07 | 71.52 | 38.67 | 76.15 | 61.62 | 59.82 | 56.03 | 80.41 | 59.71 | 54.41 | 62.34 |
| PMF+AMT-FT | ViT-small | Y | **68.80** | **77.83** | **42.90** | **79.95** | **63.77** | **63.72** | **59.06** | **85.37** | **63.87** | **57.37** | **66.26** |
| ATTNSCALE [59] | ViT-small | Y | 63.66 | 72.51 | 40.09 | 73.59 | 61.04 | 60.26 | 54.88 | 82.52 | 59.91 | 55.10 | 62.36 |
| ATTNSCALE+AMT-FT | ViT-small | Y | **68.80** | **79.43** | **42.90** | **79.95** | **63.08** | **65.66** | **59.06** | **85.37** | **64.13** | **58.24** | **66.66** |

| 5-shot | Backbone | TTF | In-domain ImageNet | Out-of-domain Omglot | Acraft | CUB | DTD | QDraw | Fungi | Flower | Sign | COCO | Avg. |
|---|---|---|---|---|---|---|---|---|---|---|---|---|---|---|
| PM [12] | ViT-small | - | 80.71 | 78.77 | 56.56 | 92.23 | 79.92 | 76.16 | 76.98 | 96.61 | 74.66 | 71.77 | 78.44 |
| StyleAdv [86] | ViT-small | - | 74.51 | 80.22 | 58.78 | 87.60 | 78.67 | 75.57 | 73.80 | 96.18 | 71.99 | 63.93 | 76.12 |
| AMT | ViT-small | - | **81.35** | **88.47** | **61.73** | **93.12** | **80.34** | **79.59** | **80.04** | **96.99** | **80.85** | **74.56** | **81.70** |
| PMF [12] | ViT-small | Y | 79.92 | 93.54 | 67.45 | 92.22 | 80.86 | 81.64 | 77.25 | 96.61 | 87.68 | 75.33 | 83.25 |
| PMF+AMT-FT | ViT-small | Y | **81.51** | **94.89** | **67.99** | **93.23** | 80.41 | **83.02** | **79.76** | **96.93** | **89.37** | **76.20** | **84.33** |
| ATTNSCALE [59] | ViT-small | Y | 79.30 | 93.48 | 69.42 | 90.49 | 81.04 | 82.66 | 77.44 | 96.51 | 89.78 | 76.48 | 83.66 |
| ATTNSCALE+AMT-FT | ViT-small | Y | **81.57** | **95.74** | **69.47** | **93.25** | 80.96 | **83.87** | **78.28** | **96.99** | **93.10** | **77.39** | **85.06** |

Table 2: **Few-shot classification clean accuracy (%) on BSCD-FSL [29] and fine-grained datasets [30] in the** 5-**way** 1-**shot and** 5-**shot settings.** We report the average accuracy and 95% confidence interval in each domain for all methods. TTF: test-time fine-tuning. Avg.: Average. **Bold** entries indicate the best for each task configuration. Rows with [†] indicate results from [86]. Other results are based on our implementations.

| 1-shot | Backbone | TTF | ChestX | ISIC | EuroSAT | CropDisease | CUB | Cars | Places | Plantae | Avg. |
|---|---|---|---|---|---|---|---|---|---|---|---|
| PM [12] | ViT-small | - | 22.74 ±0.40 | 33.72 ±0.60 | 72.94 ±0.77 | 81.04 ±0.85 | 83.53 ±0.86 | 42.10 ±0.80 | 71.66 ±0.88 | 59.04 ±0.89 | 58.35 |
| StyleAdv[†] [86] | ViT-small | - | **22.92±0.32** | 33.05±0.44 | 72.15±0.65 | 81.22±0.61 | 84.01±0.58 | 40.48±0.57 | 72.64±0.67 | 55.52±0.66 | 57.75 |
| AMT | ViT-small | - | 22.39±0.39 | **33.92 ±0.58** | **73.52±0.84** | **82.04±0.80** | **84.34 ±0.83** | **44.33 ±0.81** | **73.78±0.87** | **59.32 ±0.94** | **59.21** |
| PMF [12] | ViT-small | Y | 21.73±0.30 | 30.36±0.36 | 70.74±0.63 | 80.79±0.62 | 78.13±0.66 | 37.24±0.57 | 71.11±0.71 | 53.60±0.66 | 55.46 |
| StyleAdv-FT[†] [86] | ViT-small | Y | 22.92±0.32 | **33.99±0.46** | **74.93±0.58** | **84.11±0.57** | 84.01±0.58 | 40.48±0.57 | 72.64±0.67 | 55.52±0.66 | 58.57 |
| AMT-FT | ViT-small | Y | **23.23±0.40** | 33.95±0.63 | 73.95±0.78 | 82.04±0.8 | **84.34±0.83** | **46.06 ±0.80** | **73.83±0.89** | **59.32±0.94** | **59.59** |

| 5-shot | Backbone | TTF | ChestX | ISIC | EuroSAT | CropDisease | CUB | Cars | Places | Plantae | Avg. |
|---|---|---|---|---|---|---|---|---|---|---|---|
| PM [12] | ViT-small | - | 26.61 ±0.43 | 47.60 ±0.57 | 89.19±0.41 | 93.90±0.46 | 95.01 ±0.40 | **63.44±0.81** | 88.73±0.51 | 78.31±0.71 | 72.85 |
| StyleAdv[†] [86] | ViT-small | - | 26.97±0.33 | 47.73±0.44 | **88.57±0.34** | **94.85±0.31** | **95.82±0.27** | 61.73±0.62 | 88.33±0.40 | 75.55±0.54 | 72.44 |
| AMT | ViT-small | - | **27.54 ±0.45** | **50.22±0.63** | 88.38 ±0.48 | 94.67 ±0.40 | 94.86±0.39 | 62.94±0.82 | **88.88±0.51** | **79.32±0.7** | **73.35** |
| PMF[†] [12] | ViT-small | Y | 27.27 | 50.12 | 85.98 | 92.96 | - | - | - | - | - |
| PMF [12] | ViT-small | Y | 26.17±0.45 | 50.32±0.63 | 89.97±0.40 | 94.77 ±0.41 | 95.10±0.42 | 65.76±0.84 | 89.02±0.53 | 79.93±0.64 | 73.88 |
| StyleAdv-FT[†] [86] | ViT-small | Y | 26.97±0.33 | 51.23±0.51 | 90.12±0.33 | **95.99±0.27** | **95.82±0.27** | 66.02±0.64 | 88.33±0.40 | 78.01±0.54 | 74.06 |
| AMT-FT | ViT-small | Y | **27.54 ±0.45** | **51.56±0.68** | **90.62±0.40** | 94.67±0.40 | 95.21±0.39 | **67.18±0.79** | **89.22±0.50** | **80.36 ±0.64** | **74.54** |

on query images, we randomly sample the attack budget from attack configuration candidates used in training the robust LoRAPool. We find that this strategy improves OOD generalization but sacrifices in-domain accuracy. For the method without test-time merging, we adaptively determine the suitable LoRA in the pool based on the minimum cross-entropy loss observed in the support set. Relying solely on minimizing the cross-entropy loss on the support set can lead to overfitting, particularly on Omniglot and Traffic Sign, which have a large domain gap relative to the source domain. The degraded overall performance when removing adversarial perturbations on singular values and vectors, regardless of whether we use test-time merging strategy, verifies the role of our double-perturbation mechanism for effective robust generalization enhancement.

**Effectiveness of Adversarial Perturbation on Singular Value and Vectors**. To demonstrate the benefits of our adversarial attack strategy, we further compare AMT against the variant removing the adversarial perturbations on singular values and vectors, as shown in Figure 2. We find that our AMT can significantly amplify the magnitude of the top singular values for FFN layers (see Figure 3 in Appendix D for MHA layers). We argue that the double adversarial perturbation explicitly forces the model to focus more on the most critical components, thereby enhancing its resilience against worst-case scenarios during meta-tuning. Therefore, this improves the model's robust generalization capability, allowing it to adapt more effectively to novel downstream tasks across diverse domains.

**Different Pool Designs and Adversarial Perturbation Strategies**. As shown in Table 8 of Appendix G, the proposed robust LoRAPool with perturbation-specific parameters effectively avoids interference between different attacks and significantly enhances the out-of-domain generalization

Table 3: **Few-shot classification adversarial robust accuracy** on Meta-Dataset in the 5-way 1-shot and 5-shot settings. Adv. TTF: adversarial test-time fine-tuning.

| 1-shot | Adv. TTF | In-domain ImageNet | Out-of-domain | | | | | | | | | Avg. |
|---|---|---|---|---|---|---|---|---|---|---|---|---|
| | | | Omglot | Acraft | CUB | DTD | QDraw | Fungi | Flower | Sign | COCO | |
| PM [12] | - | 23.22 | 7.74 | 5.37 | 22.38 | 25.39 | 1.11 | 12.79 | 24.99 | 2.23 | 10.20 | 13.54 |
| StyleAdv [86] | - | 16.76 | 15.25 | 5.95 | 17.70 | 25.75 | 1.43 | 14.78 | 30.75 | 3.07 | 9.63 | 14.11 |
| AMT | - | **33.70** | **42.19** | **11.72** | **32.05** | **32.47** | **27.45** | **19.74** | **41.12** | **22.79** | **17.67** | **28.09** |
| PMF [12] | Y | 23.22 | 31.77 | 18.35 | 22.65 | 25.39 | 30.99 | 23.20 | 38.93 | 25.86 | 23.69 | 26.41 |
| AMT-FT | Y | **33.70** | **42.19** | **20.40** | **34.92** | **32.47** | **37.49** | 20.10 | **41.12** | **32.75** | 22.70 | **31.78** |

| 5-shot | Adv. TTF | In-domain ImageNet | Out-of-domain | | | | | | | | | Avg. |
|---|---|---|---|---|---|---|---|---|---|---|---|---|
| | | | Omglot | Acraft | CUB | DTD | QDraw | Fungi | Flower | Sign | COCO | |
| PM [12] | - | 36.12 | 15.29 | 8.22 | 41.93 | 40.56 | 2.53 | 23.14 | 45.27 | 4.32 | 17.75 | 23.51 |
| StyleAdv [86] | - | 29.76 | 25.35 | 8.91 | 34.06 | 40.22 | 1.98 | 23.99 | 50.66 | 5.03 | 15.89 | 23.59 |
| AMT | - | **44.69** | **65.01** | **25.10** | **58.51** | **47.82** | **41.72** | **37.70** | **68.54** | **33.41** | **29.84** | **45.23** |
| PMF [12] | Y | 36.12 | 38.43 | 21.07 | 41.93 | 40.56 | 36.51 | 26.72 | 49.83 | 29.89 | 26.47 | 34.75 |
| AMT-FT | Y | **49.62** | **68.62** | **27.26** | **59.37** | **47.82** | **60.62** | **37.70** | **72.06** | **52.70** | **37.44** | **51.32** |

Table 4: **Component ablation studies** on Meta-Dataset in the 5-way 1-shot setting. **APQ**: adversarial perturbation on query set, **APSV**: adversarial perturbation on singular values and vectors, **RLP**: Robust LoRAPool, **TTM**: test-time merging, **STr**: singular value trimming.

| APQ | APSV | RLP | TTM | STr | In-domain INet | Out-of-domain | | | | | | | | | Avg. |
|---|---|---|---|---|---|---|---|---|---|---|---|---|---|---|---|
| | | | | | | Omglot | Acraft | CUB | DTD | QDraw | Fungi | Flower | Sign | COCO | |
| ✗ | ✗ | ✗ | ✗ | ✗ | 65.07 | 59.03 | 38.13 | 76.18 | 61.56 | 57.29 | 56.03 | 80.41 | 55.17 | 54.42 | 60.35 |
| ✓ | ✗ | ✗ | ✗ | ✗ | 64.57 | 62.47 | 38.53 | 76.23 | 60.47 | 57.97 | 56.22 | 81.72 | 57.04 | 53.96 | 60.92 |
| ✓ | ✗ | ✓ | ✓ | ✗ | 65.56 | 63.92 | 39.74 | 76.06 | 61.73 | 58.64 | 55.99 | 80.93 | 56.96 | 54.28 | 61.38 |
| ✓ | ✗ | ✓ | ✓ | ✓ | 64.95 | 70.80 | 40.55 | 75.19 | 60.73 | **59.66** | 56.92 | 83.63 | 57.66 | 56.04 | 62.61 |
| ✓ | ✓ | ✓ | ✗ | ✗ | 67.95 | 62.16 | 39.13 | 79.27 | 61.77 | 58.75 | 56.59 | 79.74 | 55.45 | 54.63 | 61.54 |
| ✓ | ✓ | ✓ | ✓ | ✗ | 68.46 | 65.75 | 42.63 | 79.43 | **63.10** | 58.23 | 55.69 | 78.93 | 63.67 | 56.28 | 63.22 |
| ✓ | ✓ | ✓ | ✓ | ✓ | **68.80** | **71.95** | **42.90** | **79.95** | 62.99 | 59.62 | **59.06** | **85.37** | **63.78** | **57.14** | **65.16** |

without in-domain compromise. Furthermore, we compare with SAM [96] and the original LoRA initialization [25] in Table 9 and Table 10 of Appendix G, where the superior performance validates the efficacy of our adversarial singular value and vector perturbations in boosting the model's generalization capability. See Appendix G for more details. Moreover, in Table 16 of Appendix I, we observe that the simple pixel-level adversarial attacks can effectively simulate larger domain shifts than static data augmentation [101] and achieve comparable or superior generalization improvements compared to the learnable adversarial transformation method [102].

## 5.3 More Analysis

**Alternative Test-time Merging Strategies**. Before finalizing our test-time merging mechanism, we experimented with various design choices. The first idea involves employing a parametric linear classifier to evaluate the compatibility of LoRAs with novel tasks, similar to FLUTE [33]. To train the classifier, we input a batch of adversarial samples, each generated by attacking a different robust LoRA within the pool, to estimate which LoRA generated it. The classifier's mean output on the support set serves as the merging coefficients for a novel task. Additionally, we explored simply averaging LoRA weights or logits, similar to model soups [103]. Appendix E Table 6 compares these alternative strategies using the robust LoRAPool. We see that our AMT, with the introduced intra-class compactness and inter-class divergence criteria, achieves superior overall generalization. In contrast, the linear classifier may not accurately indicate the robustness level of adversarial perturbations based on semantic characteristics. Though logit averaging demonstrates comparable performance, it requires storing all LoRA parameters for extracting query features on each task. Our method merges the LoRAPool into the pre-trained model for adaptation on the support set, maintaining the same amount of parameters for query feature extraction as the baselines [12, 59].

**Compatibility with Other Pre-training Methods**. We evaluate the effectiveness of our AMT across different pre-training regimes on the Meta-Dataset. Previous state-of-the-art methods [12, 58, 59] employ DINO [1] pre-training on ImageNet, which utilizes the class token for self-distillation learning. We choose iBOT [2] as the representative approach using patch reconstruction as a proxy task for

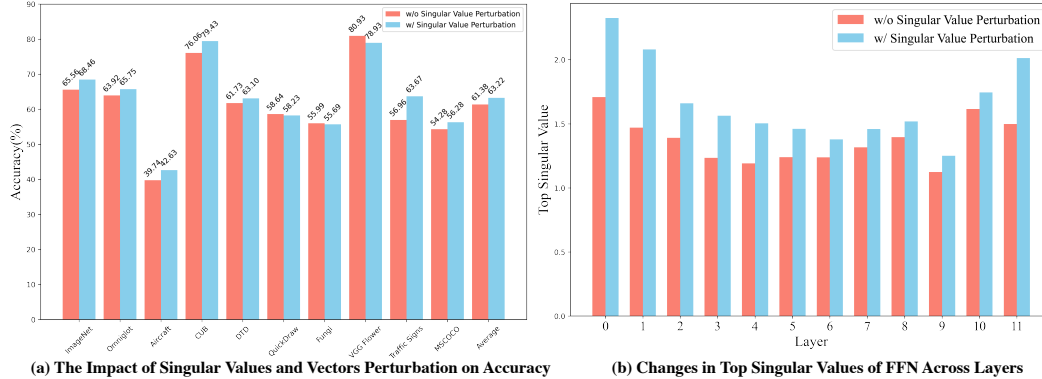

(a) The Impact of Singular Values and Vectors Perturbation on Accuracy

(b) Changes in Top Singular Values of FFN Across Layers

Figure 2: **Effectiveness of the adversarial perturbation on singular values and vectors**. The accuracy on Meta-Dataset in the 5-way 1-shot is reported.

self-supervised pre-training, DeIT [104] for supervised pre-training with strong regularizations and AdvPre [28] for adversarial pre-training. As shown in Table 7 of Appendix F, AMT achieves average performance improvements of 5.97%, 4.58%, 6.41% and 6.36% over DINO, iBOT, DeIT and AdvPre, respectively, demonstrating its effectiveness across supervised, self-supervised and robust pre-training methods. Intriguingly, we find that AMT significantly enhances the compromised in-domain clean accuracy for the adversarially robust model [28], even outperforming clean pre-trained models.

### 5.4 Comparison with Other Parameter-Efficient Fine-Tuning Methods

We compare AMT with other parameter-efficient fine-tuning methods in Table 17 of Appendix J. We observe that single Adapter-based and LoRA-based methods achieve comparable performance in adversarial meta-tuning and outperform full-model and FiLM-based meta-tuning. Besides, the superiority of the FiLM/Adapter Pool over the FiLM/Adapter signifies that our adversarial pool design contributes to the OOD performance without compromising in-domain accuracy. Also, our approach, which incorporates additional perturbation in singular values/vectors and non-parametric test-time merging mechanism utilizing the criteria (i.e., Algorithm 2) that adaptively integrates the LoRAPool into pre-trained weights, enjoys significant performance improvement over FiLM/Adapter Pool. Moreover, unlike the FLUTE-style test-time fine-tuning strategy that requires further tuning of pool components, our framework shows better compatibility with different test-time fine-tuning approaches, including LoRA tuning, full fine-tuning [12], and attention scaling [59]. More details are included in Appendix J.

## 6 Conclusions and Limitations

This paper introduces AMT employing adversarial meta-tuning to augment the robust generalization for pre-trained vision transformers. Upon generated adversarial query images at various robustness levels, we perturb the singular values and vectors to explicitly reinforce the principal components and maintain a robust LoRAPool containing perturbation-specific low-rank updates. The discriminative meta-updated components in the pool are adaptively selected and merged for customizing the model to adapt to novel tasks through a non-parametric test-time merging mechanism. Extensive experiments have demonstrated that AMT with substantial improvements in robust generalization sets new benchmarks in out-of-domain few-shot image classification tasks. Our analysis also contributes to the deeper understanding of adversarial training advancement in the few-shot setting.

Although LoRAPool has demonstrated effectiveness across different datasets and tasks, one limitation is the need for *manual* setting of the adversarial budget, particularly the size $\epsilon$, for each module. Furthermore, our exploration has been limited to adversarial budgets containing $l_\infty$ bounded perturbations, potentially restricting the ability of our method to model various types of distributional shifts. In our future work, we aim to address these limitations by expanding our exploration to include different types of adversarial perturbations and enhancing the adaptability of our method based on the specific dataset used in meta-tuning.

## Acknowledgement

This work was supported by City University of Hong Kong Project No. 9229130 and No. 9610614.

## References

[1] Mathilde Caron, Hugo Touvron, Ishan Misra, Hervé Jégou, Julien Mairal, Piotr Bojanowski, and Armand Joulin. Emerging properties in self-supervised vision transformers. In *Proceedings of the IEEE/CVF International Conference on Computer Vision*, pages 9650–9660, 2021.

[2] Jinghao Zhou, Chen Wei, Huiyu Wang, Wei Shen, Cihang Xie, Alan Yuille, and Tao Kong. ibot: Image bert pre-training with online tokenizer. *International Conference on Learning Representations*, 2022.

[3] Alec Radford, Jong Wook Kim, Chris Hallacy, Aditya Ramesh, Gabriel Goh, Sandhini Agarwal, Girish Sastry, Amanda Askell, Pamela Mishkin, Jack Clark, et al. Learning transferable visual models from natural language supervision. In *International Conference on Machine Learning*, pages 8748–8763, 2021.

[4] Maxime Oquab, Timothée Darcet, Théo Moutakanni, Huy Vo, Marc Szafraniec, Vasil Khalidov, Pierre Fernandez, Daniel Haziza, Francisco Massa, Alaaeldin El-Nouby, et al. Dinov2: Learning robust visual features without supervision. *arXiv preprint arXiv:2304.07193*, 2023.

[5] Renrui Zhang, Xiangfei Hu, Bohao Li, Siyuan Huang, Hanqiu Deng, Yu Qiao, Peng Gao, and Hongsheng Li. Prompt, generate, then cache: Cascade of foundation models makes strong few-shot learners. In *Proceedings of the IEEE/CVF Conference on Computer Vision and Pattern Recognition*, pages 15211–15222, 2023.

[6] Kun Song, Huimin Ma, Bochao Zou, Huishuai Zhang, and Weiran Huang. Fd-align: Feature discrimination alignment for fine-tuning pre-trained models in few-shot learning. *Advances in Neural Information Processing Systems*, 36, 2024.

[7] Renrui Zhang, Rongyao Fang, Wei Zhang, Peng Gao, Kunchang Li, Jifeng Dai, Yu Qiao, and Hongsheng Li. Tip-adapter: Training-free clip-adapter for better vision-language modeling. *arXiv preprint arXiv:2111.03930*, 2021.

[8] Kaiyang Zhou, Jingkang Yang, Chen Change Loy, and Ziwei Liu. Conditional prompt learning for vision-language models. In *Proceedings of the IEEE/CVF Conference on Computer Vision and Pattern Recognition*, pages 16816–16825, 2022.

[9] Kaiyang Zhou, Jingkang Yang, Chen Change Loy, and Ziwei Liu. Learning to prompt for vision-language models. *International Journal of Computer Vision (IJCV)*, 2022.

[10] Xiangyang Zhu, Renrui Zhang, Bowei He, Aojun Zhou, Dong Wang, Bin Zhao, and Peng Gao. Not all features matter: Enhancing few-shot clip with adaptive prior refinement. *arXiv preprint arXiv:2304.01195*, 2023.

[11] Markus Hiller, Rongkai Ma, Mehrtash Harandi, and Tom Drummond. Rethinking generalization in few-shot classification. *arXiv preprint arXiv:2206.07267*, 2022.

[12] Shell Xu Hu, Da Li, Jan Stühmer, Minyoung Kim, and Timothy M Hospedales. Pushing the limits of simple pipelines for few-shot learning: External data and fine-tuning make a difference. In *Proceedings of the IEEE/CVF Conference on Computer Vision and Pattern Recognition*, pages 9068–9077, 2022.

[13] Aleksander Madry, Aleksandar Makelov, Ludwig Schmidt, Dimitris Tsipras, and Adrian Vladu. Towards deep learning models resistant to adversarial attacks. In *International Conference on Learning Representations*, 2018.

[14] Micah Goldblum, Liam Fowl, and Tom Goldstein. Adversarially robust few-shot learning: A meta-learning approach. *Advances in Neural Information Processing Systems*, 33:17886–17895, 2020.

[15] Ren Wang, Kaidi Xu, Sijia Liu, Pin-Yu Chen, Tsui-Wei Weng, Chuang Gan, and Meng Wang. On fast adversarial robustness adaptation in model-agnostic meta-learning. *arXiv preprint arXiv:2102.10454*, 2021.

[16] Eleni Triantafillou, Tyler Zhu, Vincent Dumoulin, Pascal Lamblin, Utku Evci, Kelvin Xu, Ross Goroshin, Carles Gelada, Kevin Swersky, Pierre-Antoine Manzagol, et al. Meta-dataset: A dataset of datasets for learning to learn from few examples. In *International Conference on Learning Representations*, 2020.

[17] Luisa Zintgraf, Kyriacos Shiarli, Vitaly Kurin, Katja Hofmann, and Shimon Whiteson. Fast context adaptation via meta-learning. In *International Conference on Machine Learning*, pages 7693–7702, 2019.

[18] John Cai, Bill Cai, and Shen Sheng Mei. Damsl: Domain agnostic meta score-based learning. In *Proceedings of the IEEE/CVF Conference on Computer Vision and Pattern Recognition*, pages 2591–2595, 2021.

[19] Till J Bungert, Levin Kobelke, and Paul F Jaeger. Understanding silent failures in medical image classification. In *International Conference on Medical Image Computing and Computer-Assisted Intervention*, pages 400–410. Springer, 2023.

[20] Andreas Geiger, Philip Lenz, and Raquel Urtasun. Are we ready for autonomous driving? the kitti vision benchmark suite. In *Proceedings of the IEEE/CVF Conference on Computer Vision and Pattern Recognition*, pages 3354–3361, 2012.

[21] Fu Wang, Zeyu Fu, Yanghao Zhang, and Wenjie Ruan. Self-adaptive adversarial training for robust medical segmentation. In *International Conference on Medical Image Computing and Computer-Assisted Intervention*, pages 725–735. Springer, 2023.

[22] Kevin Eykholt, Ivan Evtimov, Earlence Fernandes, Bo Li, Amir Rahmati, Chaowei Xiao, Atul Prakash, Tadayoshi Kohno, and Dawn Song. Robust physical-world attacks on deep learning visual classification. In *Proceedings of the IEEE/CVF Conference on Computer Vision and Pattern Recognition*, pages 1625–1634, 2018.

[23] Qingzhao Zhang, Shengtuo Hu, Jiachen Sun, Qi Alfred Chen, and Z Morley Mao. On adversarial robustness of trajectory prediction for autonomous vehicles. In *Proceedings of the IEEE/CVF Conference on Computer Vision and Pattern Recognition*, pages 15159–15168, 2022.

[24] Behnam Neyshabur, Hanie Sedghi, and Chiyuan Zhang. What is being transferred in transfer learning? *Advances in Neural Information Processing Systems*, 33:512–523, 2020.

[25] Edward J Hu, Yelong Shen, Phillip Wallis, Zeyuan Allen-Zhu, Yuanzhi Li, Shean Wang, Lu Wang, and Weizhu Chen. Lora: Low-rank adaptation of large language models. *arXiv preprint arXiv:2106.09685*, 2021.

[26] Fanxu Meng, Zhaohui Wang, and Muhan Zhang. Pissa: Principal singular values and singular vectors adaptation of large language models. *arXiv preprint arXiv:2404.02948*, 2024.

[27] Florian Tramer and Dan Boneh. Adversarial training and robustness for multiple perturbations. *Advances in Neural Information Processing Systems*, 32, 2019.

[28] Naman Deep Singh, Francesco Croce, and Matthias Hein. Revisiting adversarial training for imagenet: Architectures, training and generalization across threat models. *Advances in Neural Information Processing Systems*, 36, 2024.

[29] Yunhui Guo, Noel C Codella, Leonid Karlinsky, James V Codella, John R Smith, Kate Saenko, Tajana Rosing, and Rogerio Feris. A broader study of cross-domain few-shot learning. In *European Conference on Computer Vision*, pages 124–141, 2020.

[30] Hung-Yu Tseng, Hsin-Ying Lee, Jia-Bin Huang, and Ming-Hsuan Yang. Cross-domain few-shot classification via learned feature-wise transformation. In *International Conference on Learning Representations*, 2020.

[31] Nikita Dvornik, Cordelia Schmid, and Julien Mairal. Selecting relevant features from a multi-domain representation for few-shot classification. In *European Conference on Computer Vision*, pages 769–786, 2020.

[32] Lu Liu, William Hamilton, Guodong Long, Jing Jiang, and Hugo Larochelle. A universal representation transformer layer for few-shot image classification. *arXiv preprint arXiv:2006.11702*, 2020.

[33] Eleni Triantafillou, Hugo Larochelle, Richard Zemel, and Vincent Dumoulin. Learning a universal template for few-shot dataset generalization. In *International Conference on Machine Learning*, pages 10424–10433, 2021.

[34] Wei-Hong Li, Xialei Liu, and Hakan Bilen. Cross-domain few-shot learning with task-specific adapters. In *Proceedings of the IEEE/CVF Conference on Computer Vision and Pattern Recognition*, pages 7161–7170, 2022.

[35] Yanbin Liu, Juho Lee, Linchao Zhu, Ling Chen, Humphrey Shi, and Yi Yang. A multi-mode modulator for multi-domain few-shot classification. In *Proceedings of the IEEE/CVF International Conference on Computer Vision*, pages 8453–8462, 2021.

[36] Xu Luo, Jing Xu, and Zenglin Xu. Channel importance matters in few-shot image classification. In *International Conference on Machine Learning*, pages 14542–14559, 2022.

[37] Chelsea Finn, Pieter Abbeel, and Sergey Levine. Model-agnostic meta-learning for fast adaptation of deep networks. In *International Conference on Machine Learning*, pages 1126–1135, 2017.

[38] Alex Nichol, Joshua Achiam, and John Schulman. On first-order meta-learning algorithms. *arXiv preprint arXiv:1803.02999*, 2018.

[39] Aniruddh Raghu, Maithra Raghu, Samy Bengio, and Oriol Vinyals. Rapid learning or feature reuse? towards understanding the effectiveness of maml. *arXiv preprint arXiv:1909.09157*, 2019.

[40] Zhenguo Li, Fengwei Zhou, Fei Chen, and Hang Li. Meta-sgd: Learning to learn quickly for few-shot learning. *arXiv preprint arXiv:1707.09835*, 2017.

[41] Eunbyung Park and Junier B Oliva. Meta-curvature. *Advances in Neural Information Processing Systems*, 32, 2019.

[42] Antreas Antoniou, Harrison Edwards, and Amos Storkey. How to train your maml. *arXiv preprint arXiv:1810.09502*, 2018.

[43] Oriol Vinyals, Charles Blundell, Timothy Lillicrap, Daan Wierstra, et al. Matching networks for one shot learning. *Advances in Neural Information Processing Systems*, 29, 2016.

[44] Jake Snell, Kevin Swersky, and Richard Zemel. Prototypical networks for few-shot learning. *Advances in Neural Information Processing Systems*, 30, 2017.

[45] Boris Oreshkin, Pau Rodríguez López, and Alexandre Lacoste. Tadam: Task dependent adaptive metric for improved few-shot learning. *Advances in Neural Information Processing Systems*, 31, 2018.

[46] Ruibing Hou, Hong Chang, Bingpeng Ma, Shiguang Shan, and Xilin Chen. Cross attention network for few-shot classification. *Advances in Neural Information Processing Systems*, 32, 2019.

[47] Carl Doersch, Ankush Gupta, and Andrew Zisserman. Crosstransformers: spatially-aware few-shot transfer. *Advances in Neural Information Processing Systems*, 33:21981–21993, 2020.

[48] James Requeima, Jonathan Gordon, John Bronskill, Sebastian Nowozin, and Richard E Turner. Fast and flexible multi-task classification using conditional neural adaptive processes. *Advances in Neural Information Processing Systems*, 32, 2019.

[49] Ethan Perez, Florian Strub, Harm De Vries, Vincent Dumoulin, and Aaron Courville. Film: Visual reasoning with a general conditioning layer. In *Proceedings of the AAAI conference on artificial intelligence*, 2018.

[50] Alexey Dosovitskiy, Lucas Beyer, Alexander Kolesnikov, Dirk Weissenborn, Xiaohua Zhai, Thomas Unterthiner, Mostafa Dehghani, Matthias Minderer, Georg Heigold, Sylvain Gelly, et al. An image is worth 16x16 words: Transformers for image recognition at scale. *arXiv preprint arXiv:2010.11929*, 2020.

[51] Ze Liu, Yutong Lin, Yue Cao, Han Hu, Yixuan Wei, Zheng Zhang, Stephen Lin, and Baining Guo. Swin transformer: Hierarchical vision transformer using shifted windows. In *Proceedings of the IEEE/CVF International Conference on Computer Vision*, pages 10012–10022, 2021.

[52] Zhengzhong Tu, Hossein Talebi, Han Zhang, Feng Yang, Peyman Milanfar, Alan Bovik, and Yinxiao Li. Maxvit: Multi-axis vision transformer. *arXiv preprint arXiv:2204.01697*, 2022.

[53] Kaiming He, Xinlei Chen, Saining Xie, Yanghao Li, Piotr Dollár, and Ross Girshick. Masked autoencoders are scalable vision learners. In *Proceedings of the IEEE/CVF Conference on Computer Vision and Pattern Recognition*, pages 16000–16009, 2022.

[54] Hangbo Bao, Li Dong, Songhao Piao, and Furu Wei. BEit: BERT pre-training of image transformers. In *International Conference on Learning Representations*, 2022.

[55] Yangji He, Weihan Liang, Dongyang Zhao, Hong-Yu Zhou, Weifeng Ge, Yizhou Yu, and Wenqiang Zhang. Attribute surrogates learning and spectral tokens pooling in transformers for few-shot learning. In *Proceedings of the IEEE/CVF Conference on Computer Vision and Pattern Recognition*, pages 9119–9129, 2022.

[56] Bowen Dong, Pan Zhou, Shuicheng Yan, and Wangmeng Zuo. Self-promoted supervision for few-shot transformer. In *European Conference on Computer Vision*, pages 329–347, 2022.

[57] Han Lin, Guangxing Han, Jiawei Ma, Shiyuan Huang, Xudong Lin, and Shih-Fu Chang. Supervised masked knowledge distillation for few-shot transformers. In *Proceedings of the IEEE/CVF Conference on Computer Vision and Pattern Recognition*, pages 19649–19659, 2023.

[58] Chengming Xu, Siqian Yang, Yabiao Wang, Zhanxiong Wang, Yanwei Fu, and Xiangyang Xue. Exploring efficient few-shot adaptation for vision transformers. *arXiv preprint arXiv:2301.02419*, 2023.

[59] Samyadeep Basu, Shell Hu, Daniela Massiceti, and Soheil Feizi. Strong baselines for parameter-efficient few-shot fine-tuning. In *Proceedings of the AAAI Conference on Artificial Intelligence*, pages 11024–11031, 2024.

[60] Christian Szegedy, Wojciech Zaremba, Ilya Sutskever, Joan Bruna, Dumitru Erhan, Ian Goodfellow, and Rob Fergus. Intriguing properties of neural networks. *arXiv preprint arXiv:1312.6199*, 2013.

[61] Ian J Goodfellow, Jonathon Shlens, and Christian Szegedy. Explaining and harnessing adversarial examples. *arXiv preprint arXiv:1412.6572*, 2014.

[62] Alexey Kurakin, Ian Goodfellow, and Samy Bengio. Adversarial machine learning at scale. *arXiv preprint arXiv:1611.01236*, 2016.

[63] Hongyang Zhang, Yaodong Yu, Jiantao Jiao, Eric Xing, Laurent El Ghaoui, and Michael Jordan. Theoretically principled trade-off between robustness and accuracy. In *International Conference on Machine Learning*, pages 7472–7482, 2019.

[64] Yisen Wang, Difan Zou, Jinfeng Yi, James Bailey, Xingjun Ma, and Quanquan Gu. Improving adversarial robustness requires revisiting misclassified examples. In *International conference on learning representations*, 2019.

[65] Tianyu Pang, Min Lin, Xiao Yang, Jun Zhu, and Shuicheng Yan. Robustness and accuracy could be reconcilable by (proper) definition. In *International Conference on Machine Learning*, pages 17258–17277, 2022.

[66] Dongxian Wu, Shu-Tao Xia, and Yisen Wang. Adversarial weight perturbation helps robust generalization. *Advances in Neural Information Processing Systems*, 33:2958–2969, 2020.

[67] Ludwig Schmidt, Shibani Santurkar, Dimitris Tsipras, Kunal Talwar, and Aleksander Madry. Adversarially robust generalization requires more data. *Advances in Neural Information Processing Systems*, 31, 2018.

[68] Dimitris Tsipras, Shibani Santurkar, Logan Engstrom, Alexander Turner, and Aleksander Madry. Robustness may be at odds with accuracy. *arXiv preprint arXiv:1805.12152*, 2018.

[69] Aditi Raghunathan, Sang Michael Xie, Fanny Yang, John Duchi, and Percy Liang. Understanding and mitigating the tradeoff between robustness and accuracy. *arXiv preprint arXiv:2002.10716*, 2020.

[70] Cihang Xie, Mingxing Tan, Boqing Gong, Jiang Wang, Alan L Yuille, and Quoc V Le. Adversarial examples improve image recognition. In *Proceedings of the IEEE/CVF Conference on Computer Vision and Pattern Recognition*, pages 819–828, 2020.

[71] Sylvestre-Alvise Rebuffi, Francesco Croce, and Sven Gowal. Revisiting adapters with adversarial training. *arXiv preprint arXiv:2210.04886*, 2022.

[72] Ruiyi Zhang, Rushi Qiang, Sai Ashish Somayajula, and Pengtao Xie. Autolora: Automatically tuning matrix ranks in low-rank adaptation based on meta learning. *arXiv preprint arXiv:2403.09113*, 2024.

[73] Sanchari Sen, Balaraman Ravindran, and Anand Raghunathan. Empir: Ensembles of mixed precision deep networks for increased robustness against adversarial attacks. *arXiv preprint arXiv:2004.10162*, 2020.

[74] Charles Herrmann, Kyle Sargent, Lu Jiang, Ramin Zabih, Huiwen Chang, Ce Liu, Dilip Krishnan, and Deqing Sun. Pyramid adversarial training improves vit performance. In *Proceedings of the IEEE/CVF Conference on Computer Vision and Pattern Recognition*, pages 13419–13429, 2022.

[75] Kaijie Zhu, Xixu Hu, Jindong Wang, Xing Xie, and Ge Yang. Improving generalization of adversarial training via robust critical fine-tuning. In *Proceedings of the IEEE/CVF International Conference on Computer Vision*, pages 4424–4434, 2023.

[76] Riccardo Volpi, Hongseok Namkoong, Ozan Sener, John C Duchi, Vittorio Murino, and Silvio Savarese. Generalizing to unseen domains via adversarial data augmentation. *Advances in Neural Information Processing Systems*, 31, 2018.

[77] Hadi Salman, Andrew Ilyas, Logan Engstrom, Ashish Kapoor, and Aleksander Madry. Do adversarially robust imagenet models transfer better? *Advances in Neural Information Processing Systems*, 33:3533–3545, 2020.

[78] Zhun Deng, Linjun Zhang, Kailas Vodrahalli, Kenji Kawaguchi, and James Y Zou. Adversarial training helps transfer learning via better representations. *Advances in Neural Information Processing Systems*, 34:25179–25191, 2021.

[79] Mingyang Yi, Lu Hou, Jiacheng Sun, Lifeng Shang, Xin Jiang, Qun Liu, and Zhiming Ma. Improved ood generalization via adversarial training and pretraing. In *International Conference on Machine Learning*, pages 11987–11997, 2021.

[80] Shiji Xin, Yifei Wang, Jingtong Su, and Yisen Wang. On the connection between invariant learning and adversarial training for out-of-distribution generalization. In *Proceedings of the AAAI Conference on Artificial Intelligence*, pages 10519–10527, 2023.

[81] Qixun Wang, Yifei Wang, Hong Zhu, and Yisen Wang. Improving out-of-distribution generalization by adversarial training with structured priors. *Advances in Neural Information Processing Systems*, 35:27140–27152, 2022.

[82] Chengxiang Yin, Jian Tang, Zhiyuan Xu, and Yanzhi Wang. Adversarial meta-learning. *arXiv preprint arXiv:1806.03316*, 2018.

[83] Junhao Dong, Yuan Wang, Jian-Huang Lai, and Xiaohua Xie. Improving adversarially robust few-shot image classification with generalizable representations. In *Proceedings of the IEEE/CVF Conference on Computer Vision and Pattern Recognition*, pages 9025–9034, 2022.

[84] Haoqing Wang and Zhi-Hong Deng. Cross-domain few-shot classification via adversarial task augmentation. *arXiv preprint*, 2021.

[85] Yanxu Hu and Andy J Ma. Adversarial feature augmentation for cross-domain few-shot classification. In *European Conference on Computer Vision*, 2022.

[86] Yuqian Fu, Yu Xie, Yanwei Fu, and Yu-Gang Jiang. Styleadv: Meta style adversarial training for cross-domain few-shot learning. In *Proceedings of the IEEE/CVF Conference on Computer Vision and Pattern Recognition*, pages 24575–24584, 2023.

[87] Xun Huang and Serge Belongie. Arbitrary style transfer in real-time with adaptive instance normalization. In *Proceedings of the IEEE/CVF International Conference on Computer Vision*, pages 1501–1510, 2017.

[88] Xiang Lisa Li and Percy Liang. Prefix-tuning: Optimizing continuous prompts for generation. *Proceedings of the 59th Annual Meeting of the Association for Computational Linguistics and the 11th International Joint Conference on Natural Language Processing (Volume 1: Long Papers)*, pages 4582–4597, 2021.

[89] Menglin Jia, Luming Tang, Bor-Chun Chen, Claire Cardie, Serge Belongie, Bharath Hariharan, and Ser-Nam Lim. Visual prompt tuning. In *European Conference on Computer Vision*, pages 709–727, 2022.

[90] Neil Houlsby, Andrei Giurgiu, Stanislaw Jastrzebski, Bruna Morrone, Quentin De Laroussilhe, Andrea Gesmundo, Mona Attariyan, and Sylvain Gelly. Parameter-efficient transfer learning for nlp. In *International Conference on Machine Learning*, pages 2790–2799, 2019.

[91] Shoufa Chen, Chongjian Ge, Zhan Tong, Jiangliu Wang, Yibing Song, Jue Wang, and Ping Luo. Adaptformer: Adapting vision transformers for scalable visual recognition. *Advances in Neural Information Processing Systems*, 35:16664–16678, 2022.

[92] Junxian He, Chunting Zhou, Xuezhe Ma, Taylor Berg-Kirkpatrick, and Graham Neubig. Towards a unified view of parameter-efficient transfer learning. In *International Conference on Learning Representations*, 2022.

[93] Mojtaba Valipour, Mehdi Rezagholizadeh, Ivan Kobyzev, and Ali Ghodsi. Dylora: Parameter efficient tuning of pre-trained models using dynamic search-free low-rank adaptation. *arXiv preprint arXiv:2210.07558*, 2022.

[94] Qingru Zhang, Minshuo Chen, Alexander Bukharin, Pengcheng He, Yu Cheng, Weizhu Chen, and Tuo Zhao. Adaptive budget allocation for parameter-efficient fine-tuning. In *International Conference on Learning Representations*, 2023.

[95] Haokun Liu, Derek Tam, Mohammed Muqeeth, Jay Mohta, Tenghao Huang, Mohit Bansal, and Colin A Raffel. Few-shot parameter-efficient fine-tuning is better and cheaper than in-context learning. *Advances in Neural Information Processing Systems*, 35:1950–1965, 2022.

[96] Pierre Foret, Ariel Kleiner, Hossein Mobahi, and Behnam Neyshabur. Sharpness-aware minimization for efficiently improving generalization. *arXiv preprint arXiv:2010.01412*, 2020.

[97] J Andrew Bagnell. Robust supervised learning. In *Proceedings of the AAAI Conference on Artificial Intelligence*, pages 714–719, 2005.

[98] Weihua Hu, Gang Niu, Issei Sato, and Masashi Sugiyama. Does distributionally robust supervised learning give robust classifiers? In *International Conference on Machine Learning*, pages 2029–2037, 2018.

[99] Prateek Yadav, Derek Tam, Leshem Choshen, Colin A Raffel, and Mohit Bansal. Ties-merging: Resolving interference when merging models. *Advances in Neural Information Processing Systems*, 36, 2024.

[100] Francesco Croce and Matthias Hein. Reliable evaluation of adversarial robustness with an ensemble of diverse parameter-free attacks. In *International Conference on Machine Learning*, pages 2206–2216, 2020.

[101] Zhenlin Xu, Deyi Liu, Junlin Yang, Colin Raffel, and Marc Niethammer. Robust and generalizable visual representation learning via random convolutions. *International Conference on Learning Representations*, 2021.

[102] Tejas Gokhale, Rushil Anirudh, Jayaraman J Thiagarajan, Bhavya Kailkhura, Chitta Baral, and Yezhou Yang. Improving diversity with adversarially learned transformations for domain generalization. In *Proceedings of the IEEE/CVF Winter Conference on Applications of Computer Vision*, 2023.

[103] Mitchell Wortsman, Gabriel Ilharco, Samir Ya Gadre, Rebecca Roelofs, Raphael Gontijo-Lopes, Ari S Morcos, Hongseok Namkoong, Ali Farhadi, Yair Carmon, Simon Kornblith, et al. Model soups: averaging weights of multiple fine-tuned models improves accuracy without increasing inference time. In *International Conference on Machine Learning*, pages 23965–23998, 2022.

[104] Hugo Touvron, Matthieu Cord, Matthijs Douze, Francisco Massa, Alexandre Sablayrolles, and Hervé Jégou. Training data-efficient image transformers & distillation through attention. In *International Conference on Machine Learning*, pages 10347–10357, 2021.

[105] Jaehoon Oh, Sungnyun Kim, Namgyu Ho, Jin-Hwa Kim, Hwanjun Song, and Se-Young Yun. Understanding cross-domain few-shot learning: An experimental study. *arXiv preprint arXiv:2202.01339*, 2022.

[106] Panagiotis Eustratiadis, Łukasz Dudziak, Da Li, and Timothy Hospedales. Neural fine-tuning search for few-shot learning. *arXiv preprint arXiv:2306.09295*, 2023.

[107] Dan Hendrycks and Thomas Dietterich. Benchmarking neural network robustness to common corruptions and perturbations. *arXiv preprint arXiv:1903.12261*, 2019.

# Mixture of Adversarial LoRAs: Boosting Robust Generalization in Meta-tuning

## *-Supplementary Material-*

## A    Algorithm of Test-time merging

The complete algorithm of our test-time merging mechanism is presented in Algorithm 2.

---
**Algorithm 2** Test-Time Merging

---
1: **Input:** Support set of meta-test task $\mathcal{S} = \{x_i^s, y_i^s\}_{i=1}^{NK}$, pre-trained residual weight matrix $W^{res}$, adaptive robust LoRAPool $\phi = [A_1 B_1, \ldots, A_P B_P]$

2: **for** $i = 1, \ldots, P$ (in parallel) **do**
3:     // Calculate the intra-class compactness
4:     $C_i = \frac{1}{NK} \sum_{s=1}^{NK} \gamma \left( \mathbf{f}_{W^{res}+A_i B_i}\left(x_s\right), \mathbf{p}_{y_s} \right)$
5:     // Calculate the inter-class divergence
6:     $V_i = \frac{1}{NK} \sum_{s=1}^{K} \sum_{\substack{c=1 \\ c \neq y_s}}^{N} \gamma \left( \mathbf{f}_{W^{res}+A_i B_i}\left(x_s\right), \mathbf{p}_c \right)$
7: **end for**
8: $\zeta_i = \frac{\text{Top}_k(\exp(-\beta(1-(\lambda C-(1-\lambda)V)))_i}{\sum_{i=1}^{k} \text{Top}_k(\exp(-\beta(1-(\lambda C-(1-\lambda)V)))_i}$
9: $W' = W_{\text{res}} + \sum_{i=1}^{P} \zeta_i A_i B_i$
10: // Singular Value Trimming
11: $\widehat{W} = \text{trim}\left(W'\right)$

---

## B    Setup for Cross-Domain Few-Shot Evaluation

### B.1    Datasets Used for Benchmarks

**Meta-Dataset** [16] is a more challenging and realistic large-scale benchmark consisting of ten image datasets including ImageNet, Omniglot, Aircraft, CUB, DTD, QuickDraw, Fungi, VGG Flower, Traffic Signs, and MSCOCO, each with specified training, validation and test splits. In this paper, we utilize the ImageNet training split as the single source domain for meta-training while employing the test splits of all datasets for meta-testing. We refer to [16] for an in-depth exploration of Meta-Dataset.

**BSCD-FSL** [29] consists of four datasets from different domains: CropDisease, EuroSAT, ISIC, and ChestX, covering plant disease images, satellite pictures, human skin lesions, and X-Ray images. We follow [29] for the dataset split.

**Fine-Grained Datasets** [30] includes four additional commonly used datasets in CD-FSL: CUB, Car, Plantae, and Places, which contain birds, cars, plant and scene images and fine-grained classes. We follow the splitting procedure of previous methods [30, 86]. We refer to [30, 105] for a more detailed description of each dataset.

### B.2    Implementation Details

We follow the pipeline delineated by PMF [12] and use the same DINO pre-training checkpoint [1] for our AMT and all baselines in main experiments. We perform adversarial meta-tuning on the ImageNet training split following the Meta-Dataset protocal [16]. The SGD optimizer with a momentum of $0.9$ and a cosine-decayed learning rate $\eta_2$ starting at $5 \times 10^{-4}$ are adopted. Training is conducted for a maximum of 30 epochs, with a 5-epoch warming-up stage. The loss trade-off coefficient $\lambda_{adv}$ is set to 6. The input image size is $128 \times 128$ as per PMF [12]. The pre-trained model is kept frozen while each LoRA is meta-updated to construct the robust LoRAPool. We use a pool of size $P = 4$ and a LoRA rank of $r = 8$, choosing the top 2 from the pool for merging at test time. Following the state-of-the-art method PMF [12], we sample five tasks from each domain as the validation set for hyperparameter selection. The adversarial query set is generated using untargeted weak and

strong patch perturbations [74] with $l_\infty$-bounded budgets $\epsilon \in \{0.01/255, 0.1/255, 6/255, 8/255\}$ in 2 steps, and a step size of $\alpha \in \left\{\frac{\epsilon}{2}, \frac{\epsilon}{10}\right\}$. The size of the neighborhood $\eta_1$ is set to $1e-4$ for adversarial perturbation on singular values and vectors. We search domain-wise hyperparameters on the validation set, including $\lambda$ in the range of $[0, 1]$, $\beta$ in the range of $[1, 12]$, and $\rho$ in the range of $[0, 1]$. The experiments were conducted on one NVIDIA A6000 GPU.

### B.3 Evaluation Metric

**Clean Few-shot Classification Accuracy**. We compute the average accuracy on the query set across 600 randomly sampled few-shot classification tasks from the test set of each dataset on three benchmarks.

**Adversarial Few-shot Classification Accuracy**. To evaluate adversarial robustness, we calculate the adversarial accuracy of the query set over 600 few-shot classification tasks. For each task, we generate adversarial examples by employing the PGD-10 attack with $l_\infty$-bounded budgets $\epsilon = 4.5/255$ and a step size $\alpha = \frac{\epsilon}{10}$ on clean images.

## C  Mathimatical Symbols

Table 5: Meaning of Math Symbols and First Appearance

| Symbol | Meaning | First Appearance |
|---|---|---|
| $\mathcal{D}_{tr}^{\text{seen}}$ | Source training dataset | Sec. 3 |
| $\mathcal{D}_{test}^{\text{unseen}}$ | Unseen target datasets | Sec. 3 |
| $\mathcal{T}$ | Task/Episode | Sec. 3 |
| $\mathcal{S}$ | Support set | Sec. 3 |
| $\mathcal{Q}$ | Query set | Sec. 3 |
| $K$ | Number of images per category in the support set | Sec. 3 |
| $N$ | Number of categories | Sec. 3 |
| $M$ | Number of images in the query set | Sec. 3 |
| $\theta$ | Pre-trained weight parameters | Sec. 3 |
| $W$ | Pre-trained weight matrix | Sec. 4.1 |
| $r$ | Rank of weight matrix | Sec. 4.1 |
| $A, B$ | Low-rank adaptors | Sec. 4.1 |
| $\epsilon$ | Perturbation budget | Sec. 4.1 |
| $\alpha$ | Perturbation step size | Sec. 4.1 |
| $\delta$ | Adversarial perturbation on images | Sec. 4.1 |
| $\delta_A, \delta_B$ | Adversarial perturbation on low-rank adaptors | Sec. 4.2 |
| $\eta_1$ | Size of the neighbourhood | Sec. 4.2 |
| $\eta_2$ | Learning rate | Sec. 4.2 |
| $U, V$ | Singular vectors | Sec. 4.2 |
| $S$ | Singular values | Sec. 4.2 |
| $W^{res}$ | Residual weight matrix | Sec. 4.2 |
| $\mathcal{L}_{CE}$ | Cross-Entropy loss | Sec. 4.2 |
| $D_{\text{KL}}$ | Kullback-Leibler divergence | Sec. 4.2 |
| $\lambda_{adv}$ | Loss function trade-off coefficient | Sec. 4.2 |
| $\phi$ | Robust LoRAPool | Sec. 4.3 |
| $\mathbf{p}_c$ | Class prototype | Sec. 4.3 |
| $C$ | Intra-class compactness | Sec. 4.3 |
| $V$ | Inter-class divergence | Sec. 4.3 |
| $\beta, \lambda$ | Smooth and balance factors | Sec. 4.3 |
| $\zeta$ | Test-time merging coefficient | Sec. 4.3 |
| $\rho$ | Singular value retaining ratio | Sec. 4.3 |
| $W'$ | Task-specific weights after test-time merging | Sec. 4.3 |
| $\widehat{W}$ | Task-specific weights after singular value trimming | Sec. 4.3 |

# D   More Analysis on Changes in Singular Values

Figure 3 shows the change in top singular values of the projection weight matrix across multi-head self-attention layers. It can be observed that our adversarial double-perturbation strategy can help the model learn to strengthen its principal components to defend against the strongest attacks during meta-tuning and thus improve generalization and robustness.

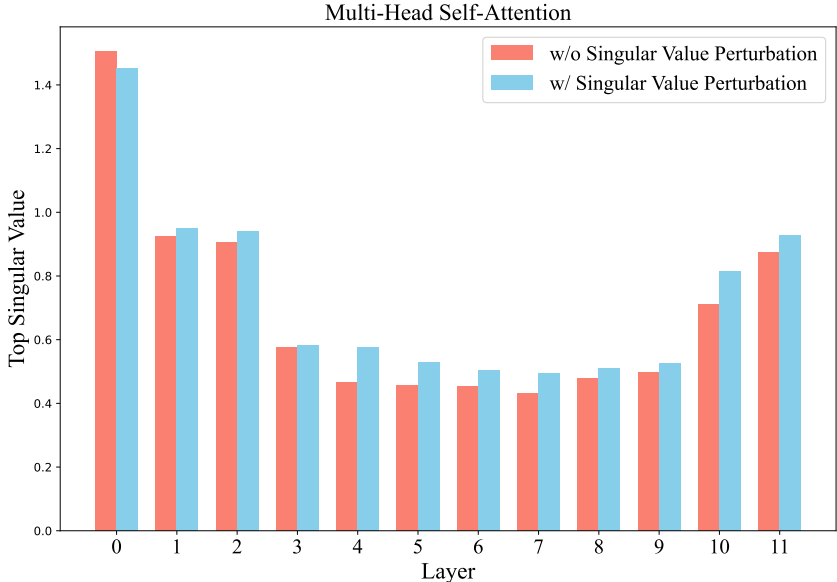

Figure 3: **Changes in top singular values of MHA across layers**

# E   Results of Alternative Test-time Merging Strategies

We compare alternative test-time merging strategies using the robust LoRAPool in Table 6. We find that our method outperforms other alternative approaches.

Table 6: **Comparison of AMT with the alternative merging strategies** on Meta-Dataset in the 5-way 1-shot setting.

| Merging Strategies | In-domain | Out-of-domain | | | | | | | | | Avg |
|---|---|---|---|---|---|---|---|---|---|---|---|
| | INet | Omglot | Acraft | CUB | DTD | QDraw | Fungi | Flower | Sign | COCO | |
| Weight Average | 63.96 | 64.49 | 40.57 | 74.23 | 59.99 | 57.36 | 55.73 | 80.50 | 59.87 | 53.04 | 60.97 |
| Logit Average | 65.84 | **66.05** | 40.70 | 78.72 | 60.79 | **59.02** | **57.42** | **82.41** | 58.41 | 55.09 | 62.44 |
| Linear classifier | 67.22 | 64.60 | 37.99 | 77.96 | 62.65 | 57.11 | 56.62 | 80.23 | 58.36 | 56.10 | 61.89 |
| AMT | **68.46** | 65.75 | **42.63** | **79.43** | **63.10** | 58.23 | 55.69 | 78.93 | **63.67** | **56.28** | **63.22** |

# F   Results With Other Pre-traing Methods

We evaluate the effectiveness of our AMT for other pre-training regimes on Meta-Dataset in Table 7. The results show that AMT achieves consistent performance improvements for various supervised, self-supervised and robust pre-training methods.

# G   More Ablation Studies of AMT

Our AMT equips a pre-trained model with a pool of adversarially meta-tuned LoRAs at varying levels of adversarial perturbation to boost the robust generalization of pre-trained models in out-of-domain

Table 7: **The compatibility of AMT with other pre-training methods** on Meta-Dataset in the 5-way 1-shot setting. All methods employ the ViT-Small architecture.

| Method | In-domain | Out-of-domain | | | | | | | | | Avg. |
|---|---|---|---|---|---|---|---|---|---|---|---|
| | INet | Omglot | Acraft | CUB | DTD | QDraw | Fungi | Flower | Sign | COCO | |
| DINO [1] | 62.91 | 59.13 | 37.11 | 73.59 | 60.67 | 57.57 | 54.88 | 78.40 | 53.62 | 53.98 | 59.19 |
| DINO+AMT | **68.80** | **71.95** | **42.90** | **79.95** | **62.99** | **59.62** | **59.06** | **85.37** | **63.78** | **57.14** | **65.16** |
| Δ | +5.89 | +12.82 | +5.79 | +6.36 | +2.32 | +2.05 | +4.18 | +6.97 | +10.16 | +3.16 | +5.97 |
| iBOT [2] | 65.09 | 61.57 | 35.40 | 70.85 | 60.36 | 57.37 | 54.47 | 78.04 | 55.00 | 55.00 | 59.32 |
| iBOT+AMT | **69.95** | **69.89** | **38.84** | **79.96** | **61.65** | **62.35** | **58.34** | **79.67** | **61.88** | **56.49** | **63.90** |
| Δ | +4.86 | +8.32 | +3.44 | +9.11 | +1.29 | +4.98 | +3.87 | +1.63 | +6.88 | +1.49 | +4.58 |
| DeIT [104] | 74.23 | 57.32 | 35.20 | 69.36 | 51.73 | 56.08 | 45.52 | 64.31 | 53.82 | 54.64 | 56.22 |
| DeIT+AMT | **81.11** | **65.50** | **38.36** | **75.80** | **56.53** | **62.16** | **53.19** | **76.09** | **58.98** | **58.57** | **62.63** |
| Δ | +6.88 | +8.18 | +3.16 | +6.44 | +4.80 | +6.08 | +7.67 | +11.78 | +5.16 | +3.93 | +6.41 |
| AdvPre [28] | 58.59 | 69.40 | 33.97 | 61.71 | 46.41 | 61.69 | 45.51 | 68.18 | 50.03 | 52.62 | 54.81 |
| AdvPre+AMT | **73.35** | **73.72** | **37.16** | **69.79** | **52.41** | **63.87** | **49.91** | **75.62** | **59.69** | **56.16** | **61.17** |
| Δ | +14.76 | +4.32 | +3.19 | +8.08 | +6.00 | +2.18 | +4.40 | +7.44 | +9.66 | +3.54 | +6.36 |

few-shot learning. Thus, we perform more ablation studies on LoRAPool design and adversarial singular value and vector perturbations.

**Different designs of robust LoRAPool**. We first adopt the uniform strategy to use the average attack strength ($\epsilon = 3.5$) of AMT's candidate configurations. Also, we develop a variant, coined random strategy, to randomly sample one attack budget $\epsilon$ for each training task from the same attack pool of candidate configurations. For a fair comparison with AMT, we meta-tune 4 LoRAs with different seeds for both the uniform and random strategies. The results are given in Table 8. We notice that the proposed robust LoRAPool with perturbation-specific parameters effectively avoids interference between different attacks and significantly enhances the out-of-domain generalization without in-domain compromise.

Table 8: **The influence of attack pool strategy** on Meta-Dataset in the 5-way 1-shot setting. **Bold** entries indicate the best for each task dataset.

| Method | In-domain | Out-of-domain | | | | | | | | | Avg. |
|---|---|---|---|---|---|---|---|---|---|---|---|
| | ImageNet | Omglot | Acraft | CUB | DTD | QDraw | Fungi | Flower | Sign | COCO | |
| Uniform LoRAPool | 63.12 | **73.28** | 42.45 | 73.59 | 59.21 | 60.22 | 53.91 | 80.77 | 59.47 | 54.04 | 62.01 |
| Random LoRAPool | 64.30 | 72.28 | **43.05** | 79.03 | 58.75 | **60.31** | 57.15 | 84.02 | 60.01 | **58.07** | 63.70 |
| Separate LoRAPool | **68.80** | 71.95 | 42.90 | **79.95** | **62.99** | 59.62 | **59.06** | **85.37** | **63.78** | 57.14 | **65.16** |

**Different perturbation strategies**. Unlike SAM [96], which employs clean examples and perturbs the weight matrices, our method applies adversarial perturbation in the spectral space, specifically targeting singular values and vectors. The results, reported in Table 9, show that AMT outperforms SAM by an average of 1.56%, substantiating the effectiveness of our adversarial perturbation strategy. We additionally compare AMT initialization against the original LoRA initialization, for which we introduce adversarial perturbations in the weight space. The superior performance, as shown in Table 10, further validates that the efficacy of our adversarial singular value and vector perturbations in boosting the model's generalization capability.

Table 9: **Comparison with SAM [96] for the perturbation input and space** on Meta-Dataset in the 5-way 1-shot setting. **Bold** entries indicate the best for each task dataset.

| Method | Input | Space | In-domain | Out-of-domain | | | | | | | | | Avg. |
|---|---|---|---|---|---|---|---|---|---|---|---|---|---|
| | | | ImageNet | Omglot | Acraft | CUB | DTD | QDraw | Fungi | Flower | Sign | COCO | |
| SAM [96] | clean | Weight | 66.50 | 71.58 | 42.79 | 79.83 | 62.15 | 59.55 | **59.18** | 80.48 | 56.73 | **57.18** | 63.60 |
| AMT | adversarial | Spectral | **68.80** | **71.95** | **42.90** | **79.95** | **62.99** | **59.62** | 59.06 | **85.37** | **63.78** | 57.14 | **65.16** |

Table 10: **The influence of adversarial perturbation space** on Meta-Dataset in the 5-way 1-shot setting. **Bold** entries indicate the best for each task dataset.

| Method | In-domain | Out-of-domain | | | | | | | | | Avg. |
|---|---|---|---|---|---|---|---|---|---|---|---|
| | ImageNet | Omglot | Acraft | CUB | DTD | QDraw | Fungi | Flower | Sign | COCO | |
| LoRA initialization | 67.26 | 71.76 | 42.90 | 79.94 | 62.20 | **60.09** | **59.31** | 80.92 | 56.70 | **57.33** | 63.84 |
| AMT initialization | **68.80** | **71.95** | 42.90 | **79.95** | **62.99** | 59.62 | 59.06 | **85.37** | **63.78** | 57.14 | **65.16** |

Table 11: **The influence of loss trade-off coefficient** $\lambda_{adv}$ on Meta-Dataset in the 5-way 1-shot setting. **Bold** entries indicate the best for each task dataset. $\star$ denotes our choice.

(a) Clean Few-shot Accuracy

| $\lambda_{adv}$ | In-domain | Out-of-domain | | | | | | | | | Avg. |
|---|---|---|---|---|---|---|---|---|---|---|---|
| | ImageNet | Omglot | Acraft | CUB | DTD | QDraw | Fungi | Flower | Sign | COCO | |
| 0 | 68.50 | 70.95 | 41.53 | 79.74 | 62.02 | 59.29 | **59.11** | 84.72 | 56.14 | 56.57 | 63.85 |
| 6$^\star$ | **68.80** | 71.95 | **42.90** | **79.95** | **62.99** | 59.62 | 59.06 | **85.37** | **63.78** | 57.14 | **65.16** |
| 8 | 67.51 | **72.23** | 42.69 | 79.02 | 62.63 | **59.97** | 58.92 | 78.10 | 61.30 | 57.17 | 63.96 |

(b) Adversarial Few-shot Accuracy

| $\lambda_{adv}$ | In-domain | Out-of-domain | | | | | | | | | Avg. |
|---|---|---|---|---|---|---|---|---|---|---|---|
| | ImageNet | Omglot | Acraft | CUB | DTD | QDraw | Fungi | Flower | Sign | COCO | |
| 0 | 22.00 | 12.58 | 5.35 | 21.15 | 22.96 | 1.74 | 10.91 | 30.66 | 1.86 | 8.77 | 13.80 |
| 6$^\star$ | **33.70** | 42.19 | 11.72 | 32.05 | 32.47 | 27.45 | 19.74 | 41.12 | 22.79 | 17.67 | 28.09 |
| 8 | 31.85 | **54.77** | **21.19** | **34.85** | **34.20** | **39.97** | **26.09** | **54.79** | **37.61** | **24.15** | **35.95** |

# H   Hyper-parameter Studies of AMT

The robust LoRAPool in AMT provides the flexibility to adjust the trade-offs between adversarial robustness and clean accuracy by modifying the pool components. In Table 11, we conduct additional experiments to highlight this benefit, using different values of $\lambda_{adv}$. The results reveal that $\lambda_{adv}$ can be used to tune LoRAPool's preference towards either clean or adversarial environments.

To analyze the impact of key hyper-parameters, we conduct experiments with various hyper-parameter values, yielding several noteworthy observations. The results in Table 12 and  13 suggest that our model is relatively insensitive to the rank of LoRA and the number of attack steps. Also, the results in Table 15 justify our choice of top-2. Furthermore, as shown in Table 14, varying the pool size $P$ and the mean and variance statistics of perturbation budget candidates $\epsilon$ demonstrates that a sufficiently diverse but large pool improves performance.

Table 12: **The influence of LoRA rank** $r$ on Meta-Dataset in the 5-way 1-shot setting. **Bold** entries indicate the best for each task dataset. $\star$ denotes our choice.

| $r$ | In-domain | Out-of-domain | | | | | | | | | Avg. |
|---|---|---|---|---|---|---|---|---|---|---|---|
| | ImageNet | Omglot | Acraft | CUB | DTD | QDraw | Fungi | Flower | Sign | COCO | |
| 4 | 68.55 | 71.94 | 42.41 | 79.69 | 62.16 | 60.91 | 59.27 | 84.38 | 63.13 | 57.72 | 65.02 |
| 8$^\star$ | **68.80** | 71.95 | 42.90 | 79.95 | 62.99 | 59.62 | 59.06 | **85.37** | **63.78** | 57.14 | **65.16** |
| 16 | 68.22 | 72.15 | **43.34** | 79.98 | 62.43 | 60.86 | 56.64 | 83.67 | 62.97 | 57.13 | 64.74 |
| 32 | 68.29 | 71.96 | 43.00 | 81.11 | 63.07 | **61.03** | **59.56** | 80.50 | 63.29 | **57.83** | 64.96 |
| 64 | 67.39 | 72.20 | 43.15 | 81.21 | 62.98 | 60.56 | 56.74 | 83.54 | 62.90 | 57.13 | 64.78 |
| 128 | 68.35 | **72.26** | 42.74 | **81.33** | **63.43** | 60.62 | 56.70 | 83.86 | 63.25 | 57.09 | 64.96 |

# I   Comparison with Other Data Augmentation Techniques

Our AMT constructs the robust LoRAPool with adaptive test-time merging to boost the robust generalization of pre-trained vision transformers in out-of-domain few-shot learning. In this context,

Table 13: **The influence of the number of attack steps** on Meta-Dataset in the 5-way 1-shot setting. **Bold** entries indicate the best for each task dataset. $^\star$ denotes our choice.

| Number | In-domain | Out-of-domain | | | | | | | | | Avg. |
|---|---|---|---|---|---|---|---|---|---|---|---|
| | ImageNet | Omglot | Acraft | CUB | DTD | QDraw | Fungi | Flower | Sign | COCO | |
| 1 | 68.06 | **73.04** | 42.22 | 79.73 | 62.22 | **60.68** | **59.21** | 82.80 | 61.69 | 56.70 | 64.64 |
| 2 $^\star$ | **68.80** | 71.95 | **42.90** | **79.95** | **62.99** | 59.62 | 59.06 | **85.37** | **63.78** | **57.14** | **65.16** |

Table 14: **The influence of the pool size $P$ and diversity** on Meta-Dataset in the 5-way 1-shot setting. **Bold** entries indicate the best for each task dataset. $^\star$ denotes our choice.

| $P$ | $\epsilon$ mean | $\epsilon$ variance | In-domain | Out-of-domain | | | | | | | | | Avg. |
|---|---|---|---|---|---|---|---|---|---|---|---|---|---|
| | | | ImageNet | Omglot | Acraft | CUB | DTD | QDraw | Fungi | Flower | Sign | COCO | |
| 1 | 3.50 | 0 | 58.80 | 67.50 | 39.63 | 64.30 | 54.16 | 59.54 | 51.87 | 78.32 | 60.44 | 50.85 | 58.54 |
| 2 | 3.05 | 8.70 | 65.54 | **72.62** | **43.39** | 76.42 | 62.54 | 59.69 | 55.81 | 82.94 | 59.51 | 56.20 | 63.48 |
| 3 | 2.04 | 7.86 | 67.60 | 72.39 | 43.14 | 79.56 | 60.68 | 60.62 | 56.86 | 85.08 | **63.88** | 56.37 | 64.62 |
| 4$^\star$ | 3.53 | 12.56 | **68.80** | 71.95 | 42.90 | 79.95 | 62.99 | 59.62 | **59.06** | **85.37** | 63.78 | 57.14 | **65.16** |
| 5 | 3.52 | 10.05 | 67.18 | 71.26 | 42.76 | **80.32** | **63.00** | **61.54** | 58.53 | 82.56 | 61.71 | **57.32** | 64.62 |
| 6 | 4.02 | 11.85 | 65.73 | 71.48 | 42.53 | 73.99 | 60.87 | 59.84 | 55.46 | 85.18 | 60.67 | 55.93 | 63.17 |

Table 15: **The influence of top-$k$** on Meta-Dataset in the 5-way 1-shot setting. **Bold** entries indicate the best for each task dataset. $^\star$ denotes our choice.

| top-$k$ | In-domain | Out-of-domain | | | | | | | | | Avg. |
|---|---|---|---|---|---|---|---|---|---|---|---|
| | ImageNet | Omglot | Acraft | CUB | DTD | QDraw | Fungi | Flower | Sign | COCO | |
| 1 | 67.70 | 70.96 | 41.59 | 77.22 | 62.15 | **61.05** | 54.58 | 81.60 | 58.24 | 55.68 | 63.08 |
| 2$^\star$ | **68.80** | **71.95** | 42.90 | 79.95 | **62.99** | 59.62 | **59.06** | **85.37** | **63.78** | 57.14 | **65.16** |
| 3 | 68.29 | 73.11 | **42.93** | **80.25** | 62.73 | 60.56 | 58.03 | 82.94 | 61.61 | **57.39** | 64.78 |
| 4 | 65.97 | 71.89 | 42.65 | 78.50 | 61.80 | 60.12 | 57.43 | 84.84 | 61.83 | 57.38 | 64.24 |

we use adversarial attacks, characterized by the size of the perturbation budget, to mimic different distributional shifts and meta-tune diverse LoRAs. The experiments in the main paper demonstrate the effectiveness of using adversarial training. In this section, we compare AMT with other data augmentation methods using a single LoRA ($P = 1$) for meta-tuning. Specifically, for ALT [102], we employ a learnable adversarial transformation network consisting of 5 convolutional layers with a kernel size of 3 and LeakyReLU activation. The adversarial learning rate was set to $5 \times 10^{-5}$, with 10 adversarial steps. For the method leveraging an attack candidate pool, we randomly select the attack budget from candidates for each training task, with $\epsilon$ values of 8/255, 6/255, 0.1/255, 0.01/255 for AMT, and step number of 1, 3, 5, 10 for ALT. As shown in Table 16, static data augmentation [101] cannot effectively simulate the large domain shifts required for robust generalization across diverse datasets (e.g., Omniglot). Compared to ALT, our AMT, utilizing only 2 steps of standard pixel-level adversarial attacks, achieves comparable or superior improvements in generalization for pre-trained vision transformers on OOD tasks.

Table 16: **Comparison with other data augmentation methods** on Meta-Dataset in the 5-way 1-shot setting. Single LoRA ($P = 1$) is used for all methods. **Bold** entries indicate the best for each task dataset.

| Method | In-domain | Out-of-domain | | | | | | | | | Avg. |
|---|---|---|---|---|---|---|---|---|---|---|---|
| | ImageNet | Omglot | Acraft | CUB | DTD | QDraw | Fungi | Flower | Sign | COCO | |
| RandConv [101] | **66.88** | 60.19 | 38.09 | 76.37 | 63.36 | 55.71 | 55.70 | 78.16 | 57.34 | **56.53** | 60.83 |
| ALT [102] + RandConv [101] | 63.98 | 63.04 | **40.28** | 75.32 | 61.25 | **58.42** | 55.86 | 81.84 | 58.55 | 53.70 | 61.22 |
| ALT [102] attack pool + RandConv [101] | 64.20 | 62.18 | 40.23 | 76.19 | **61.55** | 57.73 | 56.07 | 80.90 | **59.59** | 55.13 | 61.38 |
| AMT attack pool | 63.91 | **65.05** | 39.44 | **76.95** | 58.46 | 58.35 | **56.39** | **82.29** | 59.56 | 53.69 | **61.41** |

## J  Comparison with Other Parameter-Efficient Fine-Tuning Methods

In this section, we compare AMT with other parameter-efficient fine-tuning methods. For FiLM [49], we implement it after LN layers since there are no BN layers in ViT. We also compare Adapter [91], using the default bottleneck size of 64. The attack budget $\epsilon$ is randomly sampled from the same candidate pool as AMT for each training task. As shown in Table 17, single Adapter-based and LoRA-based methods achieve comparable performance in adversarial meta-tuning and outperform full and FiLM-based meta-tuning. Regarding the FiLM pool [49] and Adapter pool [91], we conduct additional experiments by setting the pool size to 4 and adopting the same attack pool strategy used in AMT during adversarial meta-tuning. To estimate the combination coefficients, we follow the method outlined in FLUTE [32]. Specifically, a classifier is trained in a separate stage to predict which FiLM or Adapter the input belongs to, taking as input a batch of adversarial examples generated by attacking different FiLMs or Adapters in the pool. Results show that: 1) The superiority of the FiLM/Adapter Pool over FiLM/Adapter signifies that our adversarial pool design contributes to the out-of-distribution performance without compromising in-domain accuracy. 2) Our approach, which incorporates additional perturbation in singular values/vectors and non-parametric test-time merging mechanism utilizing the criteria (i.e., Algorithm 2) that adaptively integrates the LoRAPool into pre-trained weights, enjoys significant performance improvement over FiLM/Adapter Pool. 3) Unlike the FLUTE-style test-time fine-tuning strategy that requires further tuning of pool components (either a FiLM or an adapter), our framework shows better compatibility with different test-time fine-tuning approaches, including LoRA tuning, full fine-tuning [12], and attention scaling [59].

Table 17: **Comparison with other parameter-efficient fine-tuning methods** on Meta-Dataset in the 5-way 1-shot setting.

| Adversarial Meta-tuning | Test-Time Merging | Test-Time Fine-Tuning | In-domain ImageNet | Out-of-domain | | | | | | | | | Avg. |
|---|---|---|---|---|---|---|---|---|---|---|---|---|---|
| | | | | Omglot | Acraft | CUB | DTD | QDraw | Fungi | Flower | Sign | COCO | |
| Full | - | - | 64.31 | 62.81 | 38.46 | 76.23 | **60.42** | 57.99 | 56.31 | 81.80 | 57.31 | 54.22 | 60.98 |
| Single FiLM [49] | - | - | 63.23 | 63.41 | 37.67 | 74.41 | 59.29 | 57.60 | 55.23 | 80.05 | 58.86 | **54.57** | 60.43 |
| Single Adapter [91] | - | - | **64.68** | 65.32 | 38.43 | 75.37 | 59.68 | **58.35** | 55.90 | 81.69 | 58.31 | 54.05 | 61.18 |
| Single LoRA [25] | - | - | 63.91 | 65.05 | **39.44** | **76.95** | 58.46 | 58.35 | 56.39 | 82.29 | **59.56** | 53.69 | **61.41** |
| FiLM Pool | classifier | FiLM [49] | 67.45 | 65.42 | 37.58 | 75.02 | 62.63 | 59.22 | 55.09 | 79.00 | 60.40 | 55.69 | 61.75 |
| Adapter Pool | classifier | Adapter [91] | 67.48 | 65.33 | 38.58 | **80.16** | 62.76 | 58.09 | 57.63 | 75.23 | 57.41 | 54.32 | 61.70 |
| LoRAPool | criteria | - | **68.80** | 71.95 | 42.90 | 79.95 | 62.99 | 59.62 | **59.06** | 85.37 | 63.78 | 57.14 | 65.16 |
| LoRAPool | criteria | LoRA [25] | **68.80** | 80.00 | **43.49** | 79.95 | 62.99 | 59.62 | **59.06** | 85.37 | 66.42 | 57.14 | 66.28 |
| LoRAPool | criteria | PMF [12] | **68.80** | 77.83 | 42.90 | 79.95 | **63.77** | 63.72 | **59.06** | 85.37 | 63.87 | 57.37 | 66.26 |
| LoRAPool | criteria | ATTNSCALE [59] | **68.80** | 79.43 | 42.90 | 79.95 | 63.08 | **65.66** | **59.06** | 85.37 | 64.13 | **58.24** | **66.66** |

## K  Comparison with the SOTAs in Other Settings on Meta-Dataset

In this section, we demonstrate the effectiveness of the proposed AMT under the variable-way-variable-shot setting on Meta-Dataset [16]. Table 18 presents the results of in-domain and out-of-domain few-shot classification. AMT achieves state-of-the-art overall performance in both tuning-free and test-time fine-tuning settings. Notably, it obtains **17.94%** accuracy gain ($56.91 \rightarrow 74.85$) on Omniglot tasks, which exhibit a large domain gap from the training distribution. Furthermore, with a better starting point for the test-time fine-tuning, our method shows it is promising that a plain fine-tuning approach can achieve competitive performance, even when compared to complex neural architecture search (NAS) methods, such as NFTS [106].

Table 18: **Comparison with SOTA methods on Meta-Dataset in the variable-way-variable-shot setting.** TTF: test-time fine-tuning, Avg.: Average. **Bold** entries indicate the best for each task dataset.

| Method | Backbone | TTF | In-domain INet | Out-of-domain | | | | | | | | | Avg. |
|---|---|---|---|---|---|---|---|---|---|---|---|---|---|
| | | | | Omglot | Acraft | CUB | DTD | QDraw | Fungi | Flower | Sign | COCO | |
| PM [12] | ViT-small | - | 74.69 | 56.91 | 60.5 | 85.04 | 84.21 | 61.54 | 54.78 | 94.57 | 54.21 | 57.35 | 68.38 |
| AMT | ViT-small | - | **75.72** | **74.85** | **64.87** | **87.07** | **85.06** | **66.97** | **57.58** | **95.51** | **60.55** | **58.32** | **72.65** |
| PMF [12] | ViT-small | Y | 74.69 | 80.68 | 76.78 | 85.04 | 86.63 | 71.25 | 54.78 | 94.57 | 88.33 | 62.57 | 77.53 |
| ETT [58] | ViT-small | Y | 67.4 | 78.1 | 79.9 | 85.9 | **87.6** | 71.3 | 61.8 | 96.6 | 85.1 | 62.3 | 77.6 |
| NFTS [106] | ViT-small | Y | 71.0 | 81.9 | **83.0** | 85.5 | **87.6** | 74.5 | **62.2** | 96.0 | 87.9 | 62.6 | 79.2 |
| ATTNSCALE [59] | ViT-small | Y | - | 80.9 | 78.8 | 86.7 | 85.8 | 74.4 | 59.01 | 95.9 | 91.4 | 61.9 | 79.4 |
| AMT-FT | ViT-small | Y | **75.72** | 85.54 | 80.63 | **87.07** | 86.85 | **75.65** | 57.58 | **96.26** | **92.56** | **64.49** | **80.23** |

## L   Few-shot Robustness against Natural Corruptions Under Distribution Shifts

We take a step further and investigate few-shot robustness against various types of natural visual corruptions on out-of-domain datasets, reflecting real-world conditions. We adapt ImageNet-C's methodology [107] to the Meta-Dataset benchmark by applying each category of corruption to 10 datasets. To be specific, we evaluate robustness against 15 common distortions across 4 categories (noise, blur, weather, and digital-based corruptions) with 5 severity levels. Figure 4 shows that our method AMT consistently outperforms previous counterparts across various common corruptions, demonstrating superior robustness. The ability of AMT to handle natural corruptions underscores its potential in practical applications, particularly in environments where robustness to visual corruption is critical, such as autonomous driving and medical image analysis.

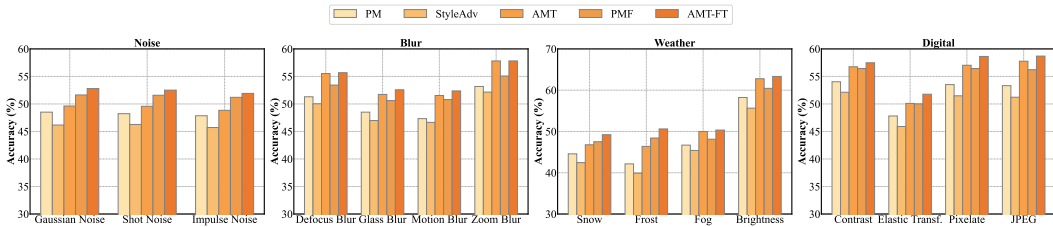

Figure 4: **The robustness averaged over Meta-Dataset datasets of different methods in the 5-way 1-shot setting.** Robustness is evaluated against 15 common distortions across four categories with varying severity levels.

## M   More Adversarial Robustness Evaluations Under Distribution Shifts

To show the effectiveness of our method for boosting adversarial robustness generalization, we conduct more robustness evaluations against unseen threat models and the stronger AutoAttack [100].

### M.1   Adversarial Robustness against AutoAttack

To measure adversarial robustness against AutoAttack [100] under distribution shifts, we ground our method and the baseline on the adversarially pre-trained ViT-Small [28]. The APGD with cross-entropy and targeted DLR loss, FAB-attack and the Square Attack are used to generate adversarial examples on 100 sampled 5-way 1-shot tasks for each dataset. We adopt $\ell_\infty$-bounded perturbations with a radius of $\epsilon_\infty = 4/255$. The results, shown in Table 19, indicate that our method AMT consistently boosts adversarial generalization across domains, even under the stronger AutoAttack, improving both in-domain and out-of-domain robust accuracy.

Table 19: **Few-shot classification adversarial robust accuracy of AutoAttack [100]** on Meta-Dataset in the 5-way 1-shot setting.

| Method | In-domain | Out-of-domain | | | | | | | | | Avg. |
|---|---|---|---|---|---|---|---|---|---|---|---|
| | ImageNet | Omglot | Acraft | CUB | DTD | QDraw | Fungi | Flower | Sign | COCO | |
| PM [12] | 29.36 | 36.52 | 3.88 | 15.06 | 14.20 | 29.80 | 3.61 | 20.48 | 10.26 | 8.26 | 17.14 |
| AMT | **39.96** | **61.48** | **8.88** | **24.04** | **23.12** | **51.48** | **11.09** | **44.76** | **23.20** | **22.00** | **31.00** |

### M.2   Adversarial Robustness against Unseen Attacks

The robust LoRAPool is constructed by adversarial meta-tuning with $\ell_\infty$-bounded perturbations on the source domain ImageNet. To evaluate the adversarial generalization against unseen attacks under distribution shifts, we use the same meta-tuned LoRAPool and employ PGD-10 attacks constrained by both $\ell_\infty$ and $\ell_2$ norms with varying perturbation budgets $\epsilon$. Specifically, we sample 600 5-way 1-shot tasks for each dataset and generate adversarial examples using 10 steps of PGD with the step size $\epsilon/10$ for $\ell_\infty$ and $\epsilon/8.5$ for $\ell_2$ attacks, respectively. The results, shown in Table 20, demonstrate that our method, AMT, significantly enhances adversarial robustness against unseen attacks under

various domains for pre-trained vision transformers. Also, compared with the previous style-based adversarial few-shoe learning method, StyleAdv [86], our AMT achieves an $\ell_\infty$ and $\ell_2$-robustness improvement of 14.76% and 13.36% in average without compromising in-domain performance.

Table 20: **Few-shot classification adversarial $\ell_\infty$, $\ell_2$-robust accuracy at different radii $\epsilon$ on** Meta-Dataset in the 5-way 1-shot setting. The evaluated models are trained on the single source domain ImageNet. **Bold** entries indicate the best for each task dataset.

| Method | In-domain | Out-of-domain | | | | | | | | | Avg. |
|---|---|---|---|---|---|---|---|---|---|---|---|
| | ImageNet | Omglot | Acraft | CUB | DTD | QDraw | Fungi | Flower | Sign | COCO | |
| $\ell_\infty(\epsilon_\infty = 8/255)$ | | | | | | | | | | | |
| PM [12] | 9.28 | 0.16 | 0.50 | 5.53 | 13.77 | 0.02 | 3.30 | 7.65 | 0.43 | 3.66 | 4.43 |
| StyleAdv [86] | 5.27 | 1.20 | 0.66 | 3.61 | 13.56 | 0.05 | 4.12 | 9.15 | 0.59 | 3.34 | 4.16 |
| AMT | **13.75** | **33.99** | **5.52** | **12.52** | **19.91** | **17.57** | **7.92** | **21.35** | **10.82** | **6.72** | **15.01** |
| $\ell_\infty(\epsilon_\infty = 6/255)$ | | | | | | | | | | | |
| PM [12] | 15.37 | 1.34 | 1.96 | 12.12 | 18.91 | 0.20 | 6.94 | 14.75 | 1.02 | 6.52 | 7.91 |
| StyleAdv [86] | 10.47 | 5.13 | 2.36 | 9.17 | 19.22 | 0.30 | 8.45 | 18.26 | 1.43 | 6.01 | 8.08 |
| AMT | **23.56** | **43.76** | **7.45** | **22.18** | **25.16** | **26.69** | **13.21** | **33.52** | **18.52** | **11.81** | **22.59** |
| $\ell_\infty(\epsilon_\infty = 4/255)$ | | | | | | | | | | | |
| PM [12] | 26.64 | 9.59 | 6.84 | 26.03 | 27.72 | 1.71 | 15.11 | 28.91 | 2.89 | 11.55 | 15.70 |
| StyleAdv [86] | 19.77 | 19.02 | 7.46 | 20.93 | 27.92 | 2.28 | 17.22 | 35.01 | 3.89 | 11.13 | 16.46 |
| AMT | **35.03** | **53.51** | **17.98** | **37.53** | **35.22** | **37.92** | **23.66** | **50.10** | **29.32** | **21.58** | **34.19** |
| $\ell_\infty(\epsilon_\infty = 2/255)$ | | | | | | | | | | | |
| PM [12] | 43.37 | 20.35 | 18.19 | 49.61 | 42.02 | 14.55 | 31.05 | 53.08 | 10.91 | 23.27 | 30.64 |
| StyleAdv [86] | 34.87 | 32.79 | 19.03 | 40.64 | 40.11 | 17.55 | 32.02 | 58.68 | 13.35 | 22.36 | 31.14 |
| AMT | **52.36** | **62.79** | **29.52** | **58.70** | **46.25** | **50.24** | **37.28** | **70.69** | **44.12** | **33.15** | **48.51** |
| $\ell_\infty(\epsilon_\infty = 1/255)$ | | | | | | | | | | | |
| PM [12] | 54.19 | 33.69 | 26.65 | 63.37 | 50.90 | 34.67 | 42.94 | 67.23 | 25.04 | 35.81 | 43.45 |
| StyleAdv [86] | 45.17 | 42.75 | 28.40 | 53.42 | 47.71 | 36.59 | 42.08 | 70.92 | 28.09 | 32.93 | 42.81 |
| AMT | **59.20** | **66.99** | **34.78** | **69.36** | **52.97** | **53.05** | **46.13** | **76.53** | **52.56** | **44.78** | **55.63** |
| $\ell_2(\epsilon_2 = 5)$ | | | | | | | | | | | |
| PM [12] | 5.26 | 0.16 | 0.17 | 1.25 | 9.27 | 0.09 | 1.04 | 3.11 | 0.22 | 1.98 | 2.26 |
| StyleAdv [86] | 2.54 | 0.69 | 0.25 | 0.84 | 8.45 | 0.08 | 1.60 | 3.87 | 0.22 | 1.76 | 2.03 |
| AMT | **7.15** | **17.60** | **2.53** | **3.11** | **14.27** | **7.88** | **3.80** | **12.07** | **4.98** | **3.69** | **7.71** |
| $\ell_2(\epsilon_2 = 3)$ | | | | | | | | | | | |
| PM [12] | 13.93 | 1.49 | 2.02 | 6.73 | 17.19 | 0.56 | 5.04 | 11.88 | 0.89 | 5.78 | 6.55 |
| StyleAdv [86] | 9.20 | 4.68 | 2.20 | 6.37 | 17.81 | 0.61 | 7.23 | 15.82 | 1.30 | 5.49 | 7.07 |
| AMT | **18.02** | **35.84** | **7.86** | **14.52** | **23.35** | **23.40** | **10.61** | **32.73** | **16.69** | **10.13** | **19.32** |
| $\ell_2(\epsilon_2 = 2)$ | | | | | | | | | | | |
| PM [12] | 24.56 | 6.54 | 6.13 | 17.75 | 25.77 | 1.92 | 12.00 | 24.38 | 2.42 | 10.63 | 13.21 |
| StyleAdv [86] | 18.01 | 14.32 | 6.69 | 16.35 | 26.62 | 2.19 | 15.41 | 31.76 | 3.39 | 10.43 | 14.52 |
| AMT | **31.52** | **47.48** | **16.25** | **27.61** | **33.76** | **35.64** | **21.68** | **48.23** | **27.75** | **21.98** | **31.19** |
| $\ell_2(\epsilon_2 = 1)$ | | | | | | | | | | | |
| PM [12] | 41.45 | 18.88 | 17.14 | 42.83 | 40.72 | 11.72 | 28.43 | 49.03 | 9.88 | 22.18 | 28.23 |
| StyleAdv [86] | 33.50 | 31.68 | 18.23 | 36.81 | 39.39 | 14.25 | 30.58 | 56.26 | 12.52 | 21.63 | 29.49 |
| AMT | **49.35** | **59.28** | **28.39** | **46.40** | **45.56** | **48.45** | **35.88** | **66.25** | **43.32** | **35.93** | **45.88** |
| $\ell_2(\epsilon_2 = 0.5)$ | | | | | | | | | | | |
| PM [12] | 52.67 | 30.98 | 25.96 | 60.04 | 50.26 | 32.08 | 41.43 | 65.27 | 24.17 | 34.94 | 41.78 |
| StyleAdv [86] | 44.95 | 41.73 | 27.74 | 51.33 | 47.34 | 34.66 | 41.28 | 69.75 | 27.62 | 32.41 | 41.88 |
| AMT | **58.78** | **65.51** | **33.06** | **65.31** | **53.15** | **52.39** | **44.93** | **77.03** | **52.29** | **44.79** | **54.72** |

